# HGDL: Heterogeneous Graph Label Distribution Learning

**Yufei Jin[†], Heng Lian[◇], Yi He[◇], Xingquan Zhu[†*]**

[†]Dept. of Elec. Eng. & Computer Sci., Florida Atlantic University, Boca Raton, FL 33431, USA
[◇]Dept. of Data Science, William & Mary, Williamsburg, VA 23185, USA
`yjin2021@fau.edu; hlian01@wm.edu; yihe@wm.edu; xzhu3@fau.edu`

## Abstract

Label Distribution Learning (LDL) has been extensively studied in IID data applications such as computer vision, thanks to its more generic setting over single-label and multi-label classification. This paper advances LDL into graph domains and aims to tackle a novel and fundamental *heterogeneous graph label distribution learning* (HGDL) problem. We argue that the graph heterogeneity reflected on node types, node attributes, and neighborhood structures can impose significant challenges for generalizing LDL onto graphs. To address the challenges, we propose a new learning framework with two key components: 1) proactive graph topology homogenization, and 2) topology and content consistency-aware graph transformer. Specifically, the former learns optimal information aggregation between meta-paths, so that the node heterogeneity can be proactively addressed prior to the succeeding embedding learning; the latter leverages an attention mechanism to learn consistency between meta-path and node attributes, allowing network topology and nodal attributes to be equally emphasized during the label distribution learning. By using KL-divergence and additional constraints, HGDL delivers an end-to-end solution for learning and predicting label distribution for nodes. Both theoretical and empirical studies substantiate the effectiveness of our HGDL approach. Our code and datasets are available at `https://github.com/Listener-Watcher/HGDL`.

## 1 Introduction

Definite supervision signals are often postulated in learning settings [3, 4]; yet, data generated from the real world tend to present inherent ambiguity, imposing challenges on assertive classifiers that predict instances into single or multiple classes. Label Distribution Learning (LDL) [5, 6, 7, 8, 9] has emerged to navigate label ambiguity by pursuing a mapping from instances to their class distributions. Each distribution quantifies the *descriptive degrees* of various classes given a specific instance.

However, the existing LDL studies mainly [10, 6, 7, 11] focus on independent and identically distributed (IID) data, such as images or texts, which do not generalize well on *graphs*. In fact, the topological structure underlying instances may provide invaluable information for label distribution learning. For example, in the task of urban planning, recent learning models have been employed to predict the point of interests (POIs) of local regions [12, 13, 14, 15]. LDL can further extend this task by providing the regional distributions over all POIs, which lends a finer-granular delineation of urban regional functionality instead of single- or multi-class classification. To wit, for a region that mixes four POIs (classes): housing, healthcare, education and worship, unlike other models assertively classify it into one or multiple POI(s), LDL model can provide insights of the functional degrees of all four POIs in this region, as shown in Figure 1. Nevertheless, existing LDL studies overlook the

---

[*]Corresponding author

urban topology, which can be rendered from, e.g., the taxi services across regions [2], missing out critical city traffic patterns that are highly correlated with regional functionalities. For instance, the regions with balanced POI distributions (e.g., $R_2$ and $R_3$) are less likely to form connections with other nodes compared to regions heavily skewed towards a single class (e.g., $R_4$), as their residents enjoy fewer needs to travel to other regions for services such as education and healthcare.

In this paper, we aim to enable and generalize the label distribution learning paradigm in networked data. Two technical challenges confront our study. First, real-world graphs are mostly heterogeneous, comprising diverse types of nodes for better expressiveness. Graph heterogeneity complicates the message-passing between nodes of a specific type (e.g., residence), as the label distributions of those nodes are influenced by their neighboring nodes that may vary in terms of types, content, and topological features. Simply leveraging node embeddings generated from message-passing for LDL will thus not work well [16, 17]. To aid, although meta-path aggregation [18] is seemingly viable, it necessitates extensive domain knowledge and expertise to craft meta-paths for each node type with respect to their label distributions; given the combinatorial number of possible meta-paths in large heterogeneous graphs, searching for the optimal meta-path for LDL is costly, laborious, and time-demanding.

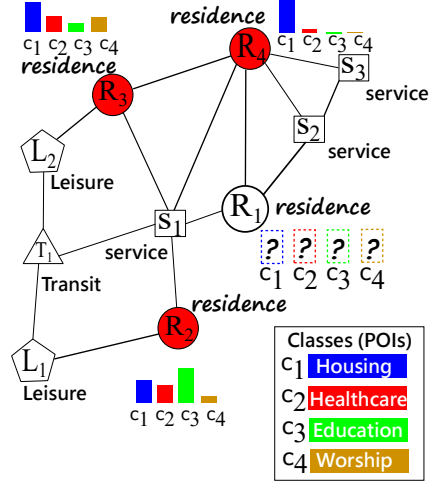

Figure 1: Motivating example of HGDL study, where each node is a local urban region [1] and edges represent taxi services [2] commuting between regions. Heterogeneous node types indicate disparate land use, including residence (R), service (S), leisure (L), and transit (T), among which R nodes are of our interest. Colored R nodes are with ground truth, which delineate their distributions over multiple point-of-interests (POIs), and each POI is deemed as a class/label. Our HGDL problem is to predict the label distribution of uncolored R nodes, enabling a precise delineation of regional urban functionality.

Second, graph topology and nodal features may suggest inconsistent label distributions, where nodes sharing similar contents are positioned far apart on the graph topology. The inconsistency is furthered in heterogeneous graphs, where nodes of the same type often connect through other intermediary types, resulting in substantial topological distances between them. Unlike traditional LDL that focuses on instance vectors only, an effective LDL model on graphs require harmonizing nodal contents with topological structures for a unified representation. The impact of distantly positioned nodes within a graph is substantially diminished, consequently steering the LDL model to prioritize individual nodal vectors, leading to compromised node representations in which the informative patterns embedded in their neighborhood structures are overlooked. Such patterns, which may significantly enhance label distribution predictions as illustrated in Figure 1, are neglected, undermining the LDL model effectiveness.

To overcome the challenges, we propose a new learning framework, coined *Heterogeneous Graph label Distribution Learning* (HGDL). Specifically, to tame the graph heterogeneity, HGDL learns the optimal graph topology of the target nodes from multiple homogeneous structures searched with various meta-path schemes through a tailored attention mechanism. The node embeddings are then generated by harmonizing the nodal features and the learned meta-path graph using a transformer architecture. A joint optimization objective is crafted based on the distance between true and predicted label distributions of the target nodes from their resultant embeddings, which unifies the learning of meta-path graph topology and the feature-topology harmonization function in an end-to-end fashion.

A key innovation of HGDL is that it changes existing heterogeneous graph learning paradigm from *reactive* (meaning that aggregation of different meta-paths are done ***after*** embedding learning from individual meta-path), to be *proactive* (meaning that aggregation are done ***before*** embedding learning). Combined with attention and transformer mechanisms to adjust individual meta-paths' interplay, and align with nodal features, HGDL deliver significantly better performance over alternatives. Our theoretical analysis assures that HGDL outperforms that of using an arbitrary meta-path graph, and HGDL's topology and feature consistency learning sparsifies network connectivity, intermediately encouraging tightens the error-bound, resulting in better model generalization.

**Specific contributions of this paper are as follows:**

1. This study pioneers the exploration of LDL problem in heterogeneous graphs. The learning problem enjoys practical implications such as for urban functionality delineation (presented in Sec 6.1) and, to our knowledge, has not yet been explored by any contemporary research.
2. We propose an end-to-end `HGDL` learning approach to jointly learn an optimal meta-path graph topology and align it with nodal features for consistent message-passing. Our approach is surprisingly simple yet effective, with its performance evidenced by theoretical underpinnings. Our approach and its analysis are presented in Sections 4 and 5, respectively.
3. Empirical study has been carried out over five graph datasets that span domains of bio-medicine, scholarly network, business network, and urban planning. Experimental results substantiate the effectiveness of our approach over rival models, documented in Section 6.

## 2  Related Work

**Label Distribution Learning (LDL)**   strives to learn a mapping from input to a distribution that profiles the descriptive degrees of classes associated with it [5, 10, 6, 7, 11]. Existing LDL methods fall into three categories, namely, problem transformation (PT), algorithm adaption (AA), and specialized algorithm (SA). PT methods transform LDL as multiple single-label learning tasks, using with label probabilities [19], and AA approaches revise mainstream learning algorithm to fit the LDL loss. SA algorithms are most commonly used because LDL learning is driven by new algorithm designs. Label correlation has been found to benefit the label distribution learning, where approaches were proposed to encode label correlation to a distance to measure the similarity of any two labels [6]. Later, low-rank approximation is used to construct label correlation matrix to capture the global label correlations [7] Instead of exploring common features for all labels, label-specific features [10] for each label are used to enhance the LDL model. Exploring feature-label and label-label correlation [9] has recently been studied in generalizable label distribution learning for cross domain learning. A Gaussian label distribution learning method [11] employs a parametric model underpinned by an optimization strategy assessing KL-divergence distance between Gaussian distributions, followed by a regression loss to normalize the KL-divergence distance. Noticing the difficulty to obtain ground-truth label distributions, Label Enhancement [20] is commonly used to recover label distributions from logical labels. Our research further push LDL to be generalized onto heterogeneous graphs, which have been overlooked by existing research. Although a recent study [21] explored using LDL in topological spaces, it focused on homogeneous graphs only and cannot work in the setting of more than one node type. Thus, the studied problem in [21] and its challenges differ from ours.

**Heterogeneous Graph Neural Networks**   have drawn extensive attention in graph learning [16, 17, 22, 18, 23, 24], because the graph heterogeneity imposes considerable challenge to model the interplay among various node types, features, labels, and network topology. Using meta-path to aggregate information from different types of nodes/edges is a common approach for heterogeneous graph learning. HetGNN [17, 18] designs graph neural networks to encode features for each type of neighbors and then aggregates neighbors' representation with respect to different types. This provides a way for GNN to deal with heterogeneous graph structures and node attributes. HAN [25] introduces attention mechanisms to heterogeneous graph learning, where attentions are applied to embedding features learned from homogeneous networks, each created from a meta-path. By doing so, attentions serve as a weighting mechanism automatically determining the importance of each meta-path for learning. Using transformers for heterogeneous networks has also been investigated recently. For example. HGT [26] designs node- and edge-type dependent parameters to characterize the heterogeneous attention over each edge, allowing this method to learn representations for different types of nodes and edges. SeHGNN [27] proposes a transformer based semantic fusion module, allowing feature fusion from different meta-paths. Our research is fundamentally different from existing work in two aspects: 1) we study LDL learning for heterogeneous networks, and 2) we propose a new way to aggregate and align information for heterogeneous network.

## 3  Preliminaries

**Notations.**   A heterogeneous graph is denoted by $G = \{V, E, X, Y\}$ associated with a node type mapping $\phi : V \mapsto \mathcal{T}^v$ and an edge type mapping $\varphi : E \mapsto \mathcal{T}^e$, with $\mathcal{T}^v$ and $\mathcal{T}^e$ the predefined and finite sets of nodes and edges, respectively, and $|\mathcal{T}^v| \geq 2$. Denote $t_\iota \in \mathcal{T}^v$ as the node type of our interest, and suppose in total $n$ nodes are of this type. Without loss of generality, we have $\phi(v_1) = \ldots = \phi(v_n) = t_\iota$, and $V_{t_\iota} = \{v_1, \ldots, v_n\} \subset V$. We deem these $n$ nodes as our *target nodes*, using a feature matrix $X \in \mathbb{R}^{n \times m}$ to describe their nodal contents, where each node contains

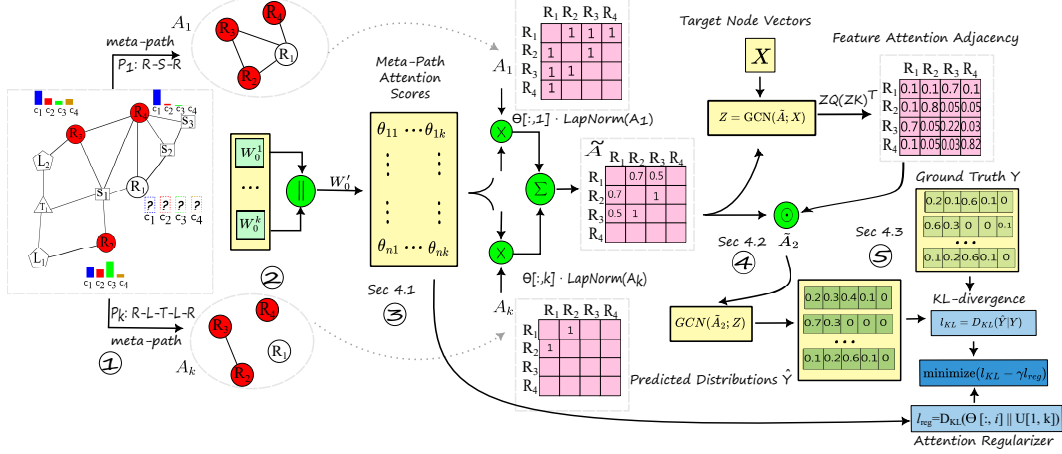

Figure 2: The proposed HGDL framework. Using $k$ meta-paths, the heterogeneous network in ① is converted to $k$ homogeneous meta-path graphs in ②. Topology homogenization in ③ proactively aggregates all $k$ meta-path graphs, through learnable weight matrix $W_0^i \in \mathbb{R}^{n \times f}$ for each graph, and finally obtain attention $\Theta \in \mathbb{R}^{n \times k}$ across all graphs. Topology and feature consistency-aware graph transformer in ④ harmonizes the local and global consistencies. The objective function in ⑤ unifies loss and regularization terms to guide nodal label distribution learning.

an $m$-dimensional feature vector. A meta-path $\mathcal{P}$ is defined as a relational sequence in form of $t_1 \xrightarrow{r_1} t_2 \ldots \xrightarrow{r_i} t_i \xrightarrow{r_{i+1}} t_{i+1} \ldots \to t_l$ (abbreviated as $\mathcal{P} = (t_1 t_2 \ldots t_i t_{i+1} \ldots t_l)$), where $(t_i t_{i+1}) \in \mathcal{T}^e$ describes the composite relation between a pair of node types. By defining a meta-path $\mathcal{P}$ with same first and last node type as the target node type, *i.e.* $t_1 = t_l = t_\iota$, we can use $\mathcal{P}$ to convert a heterogeneous network as a meta-path graph concerning only the target node type, which shall be discussed later in Section 4.1.

**Problem Statement.** In our HGDL problem, the goal is to learn a predictive mapping $\hbar : (G, X) \mapsto Y$, where $Y \in [0,1]^q$ is a distribution of descriptive labels over $q$ classes. Let $y_{i,j} \in [0,1]$ be the probability that the node $v_i$ belongs to the $j$-th class, we have $\sum_{j=1}^{q} y_{i,j} = 1$. In this work, we follow a transductive learning regime [28] to allow the ground-truth label distributions known for a subset of target nodes $V_{tr} \subset V_{t_\iota}$ during training. Our learned mapping $\hbar$ is expected to generalize well so can make accurate prediction on the remaining target nodes $V_{t_\iota} \setminus V_{tr}$.

## 4 HGDL: The Proposed Approach

**Overview.** The proposed HGDL approach comprises three key components, as illustrated in Figure 2. First, for the target nodes belonging to the node type $t_\iota$ of interest, HGDL generates multiple homogeneous meta-path graphs based on their original locations on the heterogeneous graph through meta-paths; the optimal graph topology of this node type is then learned from the homogeneous graphs via attention mechanism (*Sec 4.1*). Second, HGDL learns the embeddings of the target nodes by harmonizing the information sourced from their feature space and the learned optimal topology using a transformer-like neural architecture (*Sec 4.2*). Third, HGDL minimizes the distance between the predicted and ground-truth label distributions based on the learned node embeddings. We tailor an objective function to unify the three components into one end-to-end optimization problem, in which the optimal graph topology, the harmonization function of the feature and topological information, and the target node label distribution are jointly learned (*Sec 4.3*).

### 4.1 Optimal Graph Topology Homogenization

For a heterogenous graph, by leveraging meta-path idea, multiple different meta-path homogeneous adjacency matrix can be obtained and they can be treated as multiple sources. Graph learning is about exchanging and updating information from neighbor nodes. A proper neighbor set is therefore important for a target node to learn correct distribution. Given multiple sources, each node will have

multiple neighbor sets to choose from for updating. Traditionally, embeddings are learned for all the neighbor sets, and then aggregation over embeddings is learned. Semantics over embeddings are hard to interpret and learn compared with directly learned from different neighbor sets.

To generate a meta-path graph from the original heterogeneous graph, interactions between the meta-path and the heterogeneous graph path are used. Two nodes $v_i$ and $v_j$ are connected in the meta-path graph, if there exists a path connecting them in the heterogeneous graph, and the path follows the meta-path. Given a meta path $\mathcal{P} = (t_1 \ldots t_i t_{i+1} \ldots t_l)$, we say that a path $p = (v_1, \ldots, v_i, v_{i+1}, \ldots, v_l)$ in graph $G$ follows the meta-path $\mathcal{P}$, if $\forall i, \phi(v_i) = t_i$. Take graph in Fig. 1 as an example. Given meta-path $\mathcal{P} = (r\, s\, r)$, which defines node type $t_1 = r, t_2 = s, t_3 = r$. Path $p = (r_2, s_1, r_3)$ in the heterogeneous graph follows $\mathcal{P}$ because all nodes in the path $p$ satisfy $\phi(r_2) = t_1 = r, \phi(s_1) = t_2 = s, \phi(r_3) = t_3 = r$. Because path $p = (r_2, s_1, r_3)$ follows the meta-path $\mathcal{P}$, an edge is used to connect $r_2$ and $r_3$ in the homogeneous meta-path graph constructed from $\mathcal{P}$. Indeed, each meta-path defines a specific way of information propagation in a heterogeneous network, with resulted meta-path graph capturing unique relationships between target nodes. While defining a single meta-path is relatively easy, there often exists many meta-paths; aggregating a variety of meta-path graphs to support the downstream learning task is non-trivial.

After searching the meta-paths connecting the nodes of target type $t_\iota$, we generate a set of graphs $\mathcal{A} = \{A_1, \ldots A_k\}$, in which each adjacency $A_i \in \{0,1\}^{n \times n}$ captures the topological structure of the $i$-th meta-path-based homogeneous graph. Denoted by $A_i[p,q] = 1$ means that two target nodes $v_p$ and $v_q$, with $\phi(v_p) = \phi(v_q) = t_\iota$, are connected by a meta-path; otherwise, $A_i[p,q] = 0$. Unlike existing studies [25] that yield target node embeddings through *reactive* meta-path aggregation, where they aggregate local neighborhood information for each $A_i \in \mathcal{A}$ to capture $k$ separate meta-path topologies, our HGDL learns the optimal graph topology from $\mathcal{A}$ in a *proactive* fashion. Intuitively, HGDL learns node-level attention scores for various homogeneous graphs $A_i$, to respect the fact that the neighboring nodes may pass messages with varying importance levels in local neighborhoods, while the meta-paths walking across nodes of types other than the target $t_\iota$. Revisit the motivating example demonstrated in Fig 1 where the residence nodes are deemed as the target, the meta-path linking through the service nodes dominates, as the residence nodes are more likely to be linked through service nodes instead of Transit and Leisure nodes. Specifically, the attention scores for the nodes in every $A_i \in \mathcal{A}$ are learned in a GAT regime [29], defined as:

$$\Theta = \text{softmax}\Big(\big\|_{A_i \in \mathcal{A}} \big\{\text{LapNorm}(A_i)W_0^i\big\}W_0^{'}\Big), \quad \tilde{A} = \sum_{i=1}^{k} \Theta[:,i] \cdot \text{LapNorm}(A_i), \qquad (1)$$

where $\text{LapNorm}(A_i) = D_i^{-\frac{1}{2}}(A_i + I)D_i^{-\frac{1}{2}} \in \mathbb{R}^{n \times n}$ denotes Laplacian normalization of $A_i$, with $D_i$ being $A_i$'s degree matrix and $I$ is an identity matrix. This term mitigates the imbalanced degree distribution of the meta-path graphs. Denoted by $W_0^i$ and $W_0^{'}$ are the learnable GAT parameters, where $W_0^i \in \mathbb{R}^{n \times f}$ maps the meta-path topology of $A_i$ onto an $f$-dimensional semantic space. The operator $\|_{A_i \in \mathcal{A}}\{\cdot\}$ concatenates all $k$ resultant node embedding matrices from meta-path graphs $\mathcal{A}$, thereby producing an $\mathbb{R}^{n \times k \cdot f}$ lookup matrix, where each node is associated with a $k \cdot f$-dimensional embedding representation, and each latent $f$-dimension captures the local neighborhood structure of this node. Then, $W_0^{'} \in \mathbb{R}^{k \cdot f \times k}$ summarize the $f$-dimensional latent space into one coefficient through convex combination, resulting in attention logits, which are fed into $\text{softmax}(\cdot) = \exp(\cdot)/\sum_i \exp(\cdot)$ to yield the attention matrix $\Theta = [0,1]^{n \times k}$. Take the $i$-th column vector of $\Theta$, denoted by $\Theta[:,i] \in [0,1]^n$, we have the attention scores of $n$ target nodes for message-passing in the $i$-th meta-path graph $A_i$. We thus can deem $\tilde{A}$ in Eq.(1) as the learned optimal graph topology, which is an element-wise linear combination of $k$ meta-path topologies with the attention scores broadcast onto all $n$ nodes. Note, $\tilde{A}$ is asymmetric, namely $\tilde{A}[p,q] \neq \tilde{A}[q,p], \forall p \neq q$. This is because that the attention score of information aggregation from node $v_p$ to $v_q$ may be different from that from $v_q$ to $v_p$, as their respective local neighborhood topologies naturally differ.

## 4.2 Local Topology and Global Feature Consistency-Aware Graph Transformer

After obtaining the optimal topology $\tilde{A}$ from all meta-path graphs $A_i$, the next question is how to harmonize it with the feature information to better the target node embeddings. The benefit of such harmonization is evident. Revisiting the urban network in Fig 1, we can envision that a pair of residence nodes tend to be associated with similar embedding vectors because they enjoy two types

of consistencies: i) *local neighborhood topology*: their residents tend to travel to similar functional regions for leisure or service purposes, and ii) *global feature space*: they share similar contents such as house types and number of residing families. These local and global consistencies complement with each other, as the residence nodes having similar contents can be topologically faraway from each other on the urban network, and vice versa.

To harmonize the local and global consistencies, we are inspired by the recent graph transformers [30] and observe that the feature attention suggests a global adjacency matrix, which can be incorporated into the message-passing process. We define the graph transformer block as follows.

$$Z = \text{ReLU}(\tilde{A}XW_1), \quad \tilde{A}_2 = \text{LapNorm}\Big(\text{softmax}\big((ZQ)(ZK)^\top \odot \tilde{A}\big)\Big), \tag{2}$$

where $Z \in \mathbb{R}^{n \times h}$ is the node embeddings learned from the optimal graph topology $\tilde{A}$ through local information aggregation, parameterized by $W_1 \in \mathbb{R}^{m \times h}$. Denoted by $\tilde{A}_2 \in [0,1]^{n \times n}$ is the normalized feature attention adjacency, where $Q$ and $K \in \mathbb{R}^{h \times h}$ map the embedding matrix $Z$ onto the latent query and key spaces, respectively, such that $(ZQ)(ZK)^\top$ calculates an $n \times n$ node-level attention matrix with respect to the feature space information. Instead of normalizing the attention score by the hidden dimension $h$, we penalize the feature attention adjacency through an element-wise production $\odot$ with the optimal meta-path topology $\tilde{A}$. The intuition behind Eq. (2) is that, for each target node, it aggregates information from those neighboring nodes only if their meta-path topology and feature space are both with high attention scores. In addition, Eq. (2) functions similarly to the edge dropout [31]; in lieu of randomly removing edges, we enforce a neighbor-set intersection, where the information is only propagated from the neighbors on which the feature space and meta-path topology both agree. Such an intersection sparsity thus lowers the degree of the resultant attention adjacency, thereby uplifting the learning efficacy, which will be substantiated later in Sec 5. Finally, denoted by $H = \text{LeakyReLU}(\tilde{A}_2 ZW_2) \in \mathbb{R}^{n \times h}$ are the resultant node embeddings, capturing both local topology and global feature consistencies, which is parameterized by weight $W_2 \in \mathbb{R}^{h \times h}$.

### 4.3 An End-to-End `HGDL` Objective Function

Based on the resultant target node embeddings $H$, we can predict their label distributions as $\hat{Y} = \text{softmax}(\tilde{A}_2 HW_3) \in [0,1]^{n \times q}$, where $\hat{Y}_i = \{\hat{y}_{i,j}\}_{j=1}^q$ is the predicted label distribution of node $v_i$, among which $\hat{y}_{i,j}$ denotes its predicted probability of belonging to the class $j$. The unified objective of our `HGDL` framework is defined as follows.

$$\min_{\{W_0^i\}_{i=1}^k, W_0', W_1, W_2, W_3, K, Q} \ell_{\text{HGDL}} = D_{\text{KL}}(Y\|\hat{Y}) - \gamma \cdot \Omega,$$

$$D_{\text{KL}}(Y\|\hat{Y}) = \sum_{i=1}^n \sum_{j=1}^q y_{i,j} \cdot \log \frac{y_{i,j}}{\hat{y}_{i,j}}, \quad \Omega = \sum_{i=1}^n D_{\text{KL}}(\Theta[i,:] \| U[1,k]), \tag{3}$$

where the KL-divergence $D_{\text{KL}}(Y\|\hat{Y})$ gauges the discrepancy between the predicted and groundtruth label distributions of the target nodes [32]. The regularization term $\Omega$ gauges the distance between the attention scores of the $i$-th node across $k$ meta-path typologies (denoted by $\Theta[i,:]$) and a uniform distribution $U[1,k]$. We note the minus sign before $\Omega$, thus minimizing this term encourages a larger KL-distance, thereby avoiding the trivial uniform attention distribution (meaning that for each node, the learned attention weights from different meta-paths are encouraged to be as different as possible). $\gamma$ is a tuned parameter to balance the two terms.

## 5 Analysis

We follow the PAC-Bayes regime to analyze the theoretical performance of our `HGDL` algorithm by deriving its generalization error bound. We proceed analysis based on the meta-path graph adjacency matrices $\mathcal{A} = \{A_1, \dots A_k\}$, which are searched from the heterogeneous graph $G$. Throughout the analysis, we assume the nodal feature representations to be residing in an $\ell_2$-ball of radius $B$. We argue this a mild assumption, because in implementation we can leverage the batch-norm layers to normalize the resultant node embeddings, such that $\|\mathbf{h}_i^j\|_2 \leq B$, where $\mathbf{h}_i^j$ denotes the $i$-th node's embedding resulted from the $j$-th hidden layer.

Let $L_{\mathcal{G}}(\hbar)$ and $L_{(X,\tilde{A})}(\hbar)$ denote the *generalization risk* over a graph distribution $\mathcal{G}$ and the *empirical risk* on the target node samples and the learned meta-path topology $(X, \tilde{A})$, respectively, where $(X, \tilde{A}) \in G \overset{\text{iid}}{\sim} \mathcal{G}$. We can define:

$$L_{\mathcal{G}}(\hbar) = \mathbb{E}_{(X,\tilde{A})\sim\mathcal{G}} \mathbb{E}_{y_i\sim Y} \big[ \ell\big(\hbar(X,\tilde{A})[i], y_i\big) \big],$$

$$L_{(X,\tilde{A})}(\hbar) = \frac{1}{n} \sum_{i=1}^{n} \big[ \ell\big(\hbar(X,\tilde{A})[i], y_i\big) \big],$$

where $\ell(\cdot, \cdot)$ is a convex distance metric between two distributions that follows $|\ell(u,p) - \ell(u,q)| \leq (\sqrt{p}+1)\|p-q\|_2$, $\forall u, p, q \in \mathbb{R}^m$. Denoted by $\hbar(X,\tilde{A})[i] \in \mathbb{R}^q$ and $y_i \in [0,1]^q$ the predicted and ground-truth label distribution of the $i$-th target node, respectively. Implementing KL-divergence, we have $\ell\big(\hbar(X,\tilde{A})[i], y_i\big) = \sum_{j=1}^{q} \hbar(X,\tilde{A})[i,j] \ln(\hbar(X,\tilde{A})[i,j]/y_{i,j})$, where the predicted probability that node $i$ belongs to the $j$-th class is denoted by $\hbar(X,\tilde{A})[i,j]$. By analyzing the performance of the learned meta-path graph topology $\tilde{A}$, we find that:

**Theorem 1.** *Let $\mathbb{E}[L_{(X,A_i)}(\hbar)]$ be the empirical risk of using the $i$-th meta-path graph $A_i \in \mathcal{A}$ for label distribution prediction. With the SGD step-size $\eta$, we have*

$$L_{(X,\tilde{A})}(\hbar) \leq \min_{A_i\in\mathcal{A}} \mathbb{E}[L_{(X,A_i)}(\hbar)] + \frac{\ln k}{\eta n} + \frac{\eta}{8}.$$

**Remark 1.** Theorem 1 indicates that the empirical risk of `HGDL` is no larger than the minimum empirical risk incurred by training label distribution learner on the optimal meta-path graph, as the error bound on the RHS reduces to $\mathcal{O}(1/n)$ with constant $k$ and $\eta$. With Stochastic Gradient Descent (SGD) optimizer, larger number of target nodes $n$ will lead to more training updates over them, diminishing the $\mathcal{O}(1/n)$ bound faster. This finding substantiates the tightness of our meta-path learning strategy for the optimal graph topology $\tilde{A}$.

Due to page limits, we defer the proof of Theorem 1 and the rest analysis to the Supplement. We then analyze the generalization error bound of `HGDL` and find that:

**Theorem 2.** *Let $\hbar \in \mathcal{H} : \mathcal{X} \times \mathcal{G} \to \mathbb{R}^q$ be an $l$-layer message-passing neural network with maximum hidden dimension $k$, of which the $i$-th layer is parameterized by $W_i$. Then for any $\delta, \gamma, B > 0$ and $l > 1$, with probability at least $1 - \delta$ we have*

$$L_{\mathcal{G}}(\hbar) - L_{(X,\tilde{A})}(\hbar) \leq \frac{2(\sqrt{2q}+\sqrt{2})q}{\sqrt{n}} \max_{i\in[n],j\in[l]} \left\| \mathbf{h}_i^j \right\|_2$$

$$+ 3b\sqrt{\frac{\log 2/\delta}{2n}} + \mathcal{O}\left( \sqrt{\frac{B^2 d^{l-1} l^2 k \log(lk)\mathcal{D}(W_i) + \log\frac{nl}{\delta}}{\gamma^2 n}} \right),$$

*where $\mathcal{D}(W_i) = \prod_{i=1}^{l} \|W_i\|_2^2 \cdot \sum_{i=1}^{l} \big( \|W_i\|_F^2 / \|W_i\|_2^2 \big)$ bounds the hypothesis space and $b$ is a constant.*

**Remark 2.** We remark several key observations from Theorems 1 and 2. First, the generalization capability of the algorithm is negatively impacted by a higher dimensional label space $q$. Second, the robustness of `HGDL` decreases with larger $B$ values, which gauges the magnitude of perturbations thus the inherent high data variance. Third, as the graph neural network architecture becomes deeper (larger $l$) or wider (larger $k$), the generalization risk increases, suggesting the potential risk of model overfitting. Forth, with larger $n$, the generalization error bound diminishes, which indicates that the meta-path topology can be better delineated with an increased number of target nodes on the graph.

**Remark 3.** By combining Theorems 1 and 2, we observe that the generalization error bound of `HGDL` using $\tilde{A}$ outperforms that of using an arbitrary meta-path graph $A_i$. Further, it is easy to verify that the maximum degree of $\tilde{A}$, denoted by $d$, is smaller than that of $A_i$, denoted by $d_i$, *i.e.*, $d \leq d_i$, $\forall i \in [k]$. This rationalizes our graph transformer design in Sec 4.2, where the enforced topology and feature consistency in Eq. (2) sparsifies network connectivity, thereby intermediately encourages better model generalization.

# 6 Experiments

## 6.1 Experiment Setup

| Dataset | # node type | # nodes | # edges | # features | # labels |
|---|---|---|---|---|---|
| DRUG | 4 | 40,786 | 1,737,890 | 191 | 28 |
| ACM | 5 | 20,200 | 104,976 | 1,903 | 14 |
| DBLP | 4 | 27,325 | 148,246 | 8,920 | 4 |
| YELP | 4 | 8,052,542 | 7,905,197 | 19 | 9 |
| URBAN | 4 | 1,434 | 42,857 | 155 | 10 |

Table 1: Summary of dataset statistics.

**Benchmark Datasets** To our best knowledge, no heterogeneous graph dataset with *ground-true label distributions* currently exists. To level the comparison study, we prepare five datasets with ground-truth node label distributions using existing heterogeneous graphs, including DBLP [33], ACM [33], YELP, DRUG [34], and URBAN [1]. Table 1 summarizes the data statistics. A detailed description on the dataset creation and preprocessing, as well as their domain and label semantic meanings, has been deferred to the Supplement B due to space limitation.

**Compared Models** In total six competitors are identified for comparative study. As no model directly resolving the HGDL problem exists, we employ the state-of-the-art heterogeneous graph neural networks and integrate them KL-divergence loss to learn label distributions of nodes. They include: 1) $GCN_{KL}$: A baseline that uses graph constructed from each meta-path to train a vanilla GCN [35], using KL-divergence as loss function, and reports the best meta-path result; 2) $HAN_{KL}$: This baseline uses HAN [25] to integrate embedding from different meta-paths; and 3) $SeHGNN_{KL}$: This baseline uses SeHGNN [27], a transformer based approach, to aggregate meta-paths embedding with KL-divergence loss function. For ablation study, we further include three variants reduced from our proposed HGDL method, which include: 4) $HGDL_{\neg TH}$: it removes HGDL's topology homogenization (Sec 4.1), which learns embedding from each meta-path graph and reports the best meta-path result; 5) $HGDL_{\neg transformer}$: it uses GCN instead of the transformer (Sec 4.2) to learn embedding to validate HGDL's transformer for embedding learning; and 6) $HGDL_{ED}$: it replaces HGDL's topology and feature consistence-aware graph transformer (Sec. 4.2) by using a random edge dropout method [31].

**Evaluation Metrics** To measure the discrepancy between two distributions, i.e., the predicted and true label distributions of target nodes, we identify six metrics: Cosine Distance (COD), Canberra Distance (CAD), Chebyshev Distance (CHD), Clark Distance (CLD); Intersection Score (IND), and Kullback-Leibler Divergence (KL). Their definitions and calculations are deferred to Supplement B.

| Dataset | Model | COD↓ | CAD↓ | CHD↓ | CLD↓ | IND↑ | KL↓ | Win/Tie/Lose |
|---|---|---|---|---|---|---|---|---|
| DRUG | $GCN_{KL}$ | $0.220_{\pm.025}$ | $9.209_{\pm.740}$ | $0.245_{\pm.031}$ | $1.963_{\pm.134}$ | $0.676_{\pm.029}$ | $0.484_{\pm.075}$ | 4/2/0 |
| | $HAN_{KL}$ | $0.279_{\pm.023}$ | $10.084_{\pm.823}$ | $0.268_{\pm.027}$ | $2.155_{\pm.148}$ | $0.632_{\pm.025}$ | $0.579_{\pm.055}$ | 6/0/0 |
| | $SeHGNN_{KL}$ | $0.286_{\pm.018}$ | $10.178_{\pm.781}$ | $0.267_{\pm.020}$ | $2.166_{\pm.143}$ | $0.640_{\pm.016}$ | $0.600_{\pm.059}$ | 6/0/0 |
| | HGDL | $\mathbf{0.168}_{\pm.019}$ | $\mathbf{9.179}_{\pm.574}$ | $\mathbf{0.217}_{\pm.017}$ | $\mathbf{1.957}_{\pm.114}$ | $\mathbf{0.710}_{\pm.020}$ | $\mathbf{0.392}_{\pm.044}$ | – |
| | $HGDL_{\neg transformer}$ | $0.199_{\pm.014}$ | $9.371_{\pm.679}$ | $0.235_{\pm.017}$ | $2.004_{\pm.137}$ | $0.687_{\pm.021}$ | $0.492_{\pm.059}$ | 4/2/0 |
| | $HGDL_{\neg TH}$ | $0.212_{\pm.023}$ | $9.510_{\pm.602}$ | $0.240_{\pm.018}$ | $2.029_{\pm.110}$ | $0.671_{\pm.020}$ | $0.462_{\pm.050}$ | 4/2/0 |
| | $HGDL_{ED}$ | $0.204_{\pm.026}$ | $9.602_{\pm.882}$ | $0.239_{\pm.028}$ | $2.040_{\pm.162}$ | $0.681_{\pm.030}$ | $0.574_{\pm.085}$ | 4/2/0 |
| ACM | $GCN_{KL}$ | $0.217_{\pm.007}$ | $13.101_{\pm.014}$ | $0.337_{\pm.012}$ | $3.527_{\pm.057}$ | $0.652_{\pm.013}$ | $0.842_{\pm.072}$ | 5/1/0 |
| | $HAN_{KL}$ | $0.212_{\pm.005}$ | $13.114_{\pm.009}$ | $0.371_{\pm.008}$ | $3.485_{\pm.023}$ | $0.618_{\pm.008}$ | $0.765_{\pm.015}$ | 5/1/0 |
| | $SeHGNN_{KL}$ | $0.247_{\pm.061}$ | $13.141_{\pm.052}$ | $0.371_{\pm.082}$ | $3.492_{\pm.061}$ | $0.617_{\pm.090}$ | $0.924_{\pm.166}$ | 4/2/0 |
| | HGDL | $\mathbf{0.203}_{\pm.004}$ | $\mathbf{13.098}_{\pm.006}$ | $0.351_{\pm.004}$ | $3.408_{\pm.035}$ | $\mathbf{0.637}_{\pm.004}$ | $\mathbf{0.753}_{\pm.025}$ | – |
| | $HGDL_{\neg transformer}$ | $0.211_{\pm.008}$ | $13.099_{\pm.010}$ | $0.358_{\pm.011}$ | $3.403_{\pm.027}$ | $0.630_{\pm.012}$ | $0.777_{\pm.031}$ | 1/5/0 |
| | $HGDL_{\neg TH}$ | $0.223_{\pm.006}$ | $13.130_{\pm.015}$ | $0.361_{\pm.019}$ | $3.423_{\pm.022}$ | $0.631_{\pm.020}$ | $0.879_{\pm.034}$ | 3/3/0 |
| | $HGDL_{ED}$ | $0.216_{\pm.007}$ | $13.106_{\pm.005}$ | $0.364_{\pm.006}$ | $\mathbf{3.375}_{\pm.041}$ | $0.624_{\pm.007}$ | $0.801_{\pm.020}$ | 5/1/0 |
| DBLP | $GCN_{KL}$ | $0.031_{\pm.004}$ | $2.852_{\pm.009}$ | $0.091_{\pm.006}$ | $1.647_{\pm.002}$ | $0.908_{\pm.006}$ | $0.114_{\pm.011}$ | 6/0/0 |
| | $HAN_{KL}$ | $0.025_{\pm.002}$ | $2.819_{\pm.012}$ | $0.071_{\pm.004}$ | $1.633_{\pm.007}$ | $0.929_{\pm.004}$ | $0.082_{\pm.008}$ | 5/1/0 |
| | $SeHGNN_{KL}$ | $0.086_{\pm.140}$ | $2.887_{\pm.170}$ | $0.155_{\pm.208}$ | $\mathbf{1.624}_{\pm.049}$ | $0.842_{\pm.214}$ | $0.252_{\pm.397}$ | 0/6/0 |
| | HGDL | $\mathbf{0.019}_{\pm.002}$ | $\mathbf{2.796}_{\pm.014}$ | $\mathbf{0.057}_{\pm.005}$ | $1.633_{\pm.005}$ | $\mathbf{0.943}_{\pm.005}$ | $\mathbf{0.057}_{\pm.011}$ | – |
| | $HGDL_{\neg transformer}$ | $0.025_{\pm.002}$ | $2.828_{\pm.004}$ | $0.074_{\pm.005}$ | $1.642_{\pm.001}$ | $0.925_{\pm.005}$ | $0.090_{\pm.008}$ | 6/0/0 |
| | $HGDL_{\neg TH}$ | $0.020_{\pm.002}$ | $2.808_{\pm.005}$ | $0.062_{\pm.005}$ | $1.637_{\pm.005}$ | $0.937_{\pm.005}$ | $0.070_{\pm.011}$ | 3/3/0 |
| | $HGDL_{ED}$ | $0.023_{\pm.001}$ | $2.819_{\pm.005}$ | $0.070_{\pm.003}$ | $1.639_{\pm.003}$ | $0.929_{\pm.003}$ | $0.082_{\pm.006}$ | 6/0/0 |
| YELP | $GCN_{KL}$ | $\mathbf{0.342}_{\pm.014}$ | $7.180_{\pm.125}$ | $0.458_{\pm.015}$ | $2.558_{\pm.031}$ | $0.456_{\pm.016}$ | $1.037_{\pm.044}$ | 0/6/0 |
| | $HAN_{KL}$ | $0.453_{\pm.163}$ | $5.894_{\pm1.808}$ | $0.569_{\pm.158}$ | $\mathbf{2.226}_{\pm.461}$ | $0.379_{\pm.118}$ | $5.832_{\pm6.577}$ | 3/2/1 |
| | $SeHGNN_{KL}$ | $0.404_{\pm.106}$ | $6.298_{\pm1.757}$ | $0.522_{\pm.111}$ | $2.343_{\pm.438}$ | $0.413_{\pm.078}$ | $3.993_{\pm5.426}$ | 0/6/0 |
| | HGDL | $\mathbf{0.342}_{\pm.015}$ | $7.177_{\pm.128}$ | $\mathbf{0.457}_{\pm.016}$ | $2.558_{\pm.031}$ | $\mathbf{0.459}_{\pm.015}$ | $\mathbf{1.034}_{\pm.041}$ | – |
| | $HGDL_{\neg transformer}$ | $\mathbf{0.342}_{\pm.014}$ | $7.175_{\pm.128}$ | $0.458_{\pm.016}$ | $2.557_{\pm.031}$ | $0.458_{\pm.015}$ | $1.035_{\pm.039}$ | 0/6/0 |
| | $HGDL_{\neg TH}$ | $\mathbf{0.342}_{\pm.015}$ | $\mathbf{7.173}_{\pm.126}$ | $0.458_{\pm.017}$ | $2.556_{\pm.031}$ | $0.458_{\pm.016}$ | $1.046_{\pm.051}$ | 0/6/0 |
| | $HGDL_{ED}$ | $0.348_{\pm.021}$ | $7.221_{\pm.174}$ | $0.463_{\pm.020}$ | $2.565_{\pm.038}$ | $0.453_{\pm.019}$ | $1.070_{\pm.078}$ | 0/6/0 |
| URBAN | $GCN_{KL}$ | $0.485_{\pm.025}$ | $8.318_{\pm.037}$ | $0.536_{\pm.014}$ | $\mathbf{2.773}_{\pm.009}$ | $0.331_{\pm.011}$ | $1.386_{\pm.069}$ | 2/4/0 |
| | $HAN_{KL}$ | $0.497_{\pm.017}$ | $8.337_{\pm.029}$ | $0.538_{\pm.010}$ | $2.777_{\pm.008}$ | $0.326_{\pm.005}$ | $1.407_{\pm.054}$ | 4/2/0 |
| | $SeHGNN_{KL}$ | $0.497_{\pm.019}$ | $8.336_{\pm.029}$ | $0.537_{\pm.011}$ | $2.776_{\pm.008}$ | $0.327_{\pm.006}$ | $1.409_{\pm.061}$ | 4/2/0 |
| | HGDL | $\mathbf{0.467}_{\pm.023}$ | $\mathbf{8.315}_{\pm.041}$ | $\mathbf{0.517}_{\pm.012}$ | $2.775_{\pm.009}$ | $\mathbf{0.356}_{\pm.010}$ | $\mathbf{1.340}_{\pm.065}$ | – |
| | $HGDL_{\neg transformer}$ | $0.497_{\pm.013}$ | $8.340_{\pm.032}$ | $0.537_{\pm.006}$ | $2.777_{\pm.008}$ | $0.325_{\pm.006}$ | $1.409_{\pm.041}$ | 4/2/0 |
| | $HGDL_{\neg TH}$ | $0.500_{\pm.016}$ | $8.338_{\pm.028}$ | $0.540_{\pm.010}$ | $2.776_{\pm.008}$ | $0.321_{\pm.004}$ | $1.414_{\pm.046}$ | 4/2/0 |
| | $HGDL_{ED}$ | $0.481_{\pm.028}$ | $8.325_{\pm.031}$ | $0.527_{\pm.012}$ | $2.775_{\pm.007}$ | $0.340_{\pm.017}$ | $1.375_{\pm.067}$ | 1/5/0 |

Table 2: Mean ± standard deviation results of seven models on five datasets. Best results are bold, and ↑ (or ↓) indicates the higher (or lower) the better. Results are taken from 5 repeats. The win/tie/loss counts are suggested by the *paired t-test* at 90% confidence level.

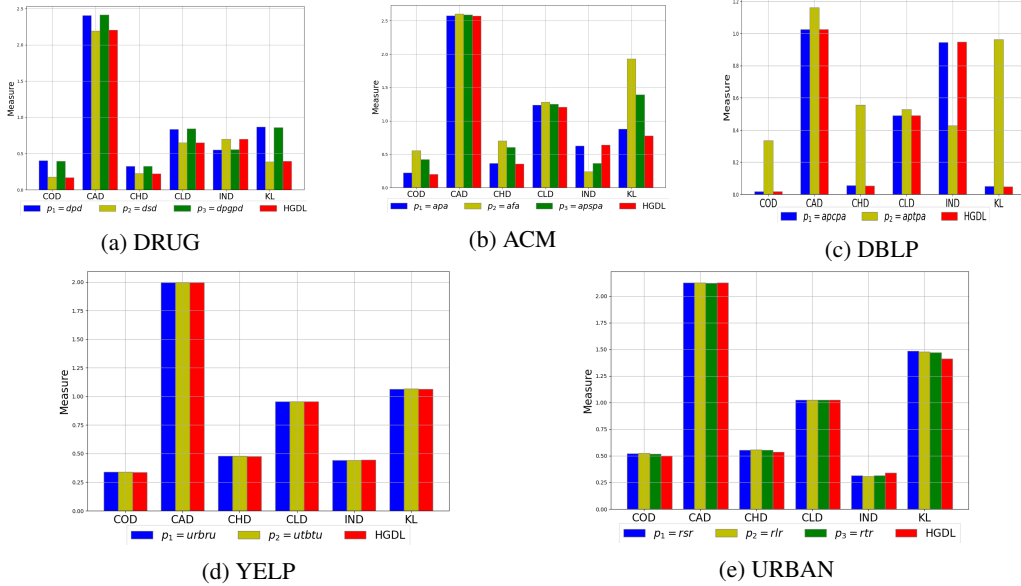

(a) DRUG           (b) ACM           (c) DBLP

(d) YELP           (e) URBAN

Figure 3: Comparisons between HGDL *vs.* results from a single meta-path (CAD and CLD are calculated in natural log for better visualization) for five datasets.

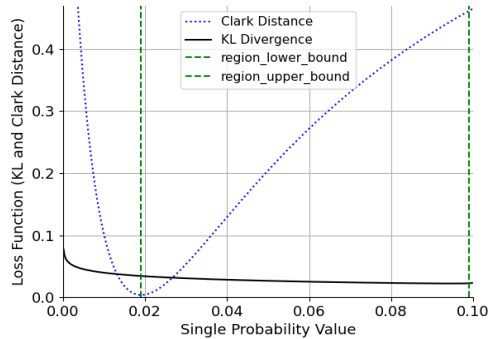

Figure 4: KL and CLD tradeoff function example. The estimated probability distribution is $[x_1, x_2, 0.9]$ and true probability distribution is $[0.05, 0.05, 0.9]$, with $x_1 + x_2 = 0.1$. Horizontal axis is the $x_1$ value and vertical axis is the loss for both CLD and KL divergence. Green dashed lines cover the tradeoff region where the CLD loss monotonically increases and KL-divergence decreases.

## 6.2 Results

Table 2 summarizes results of all methods. Overall, our HGDL wins in 99 out of 180 settings, among which on average 20 out of 30 settings excel in COD, CHD, IND, and KL metrics, 11 out of 30 in CAD, and 8 out of 30 in CLD. On DURG, ACM, DBLP, and URBAN datasets, HGDL outperforms its competitors in 83% settings in COD, CHD, IND, and KL metrics 46% in CAD, and 20% in CLD. Beyond its overall better comparative performance, we make the following observation on HGDL. First, $HAN_{KL}$ and $SeHGNN_{KL}$ achieve better performance on YELP and DBLP dataset for CAD and CLD metrics, but not on the other datasets and metrics. This shows that existing meta-path based methods cannot learn distribution prediction well. In general, these models show similar performance on YELP dataset. We hypothesize that this is due to the lack of rich feature information on YELP, of which the dimension of nodal features is 19 which is minimum across all datasets. Second, HGDL achieves the best results in KL-divergence by a large margin across all settings. On average, HGDL have a 15% improvement compared with the second best result across all datasets in KL-divergence. Given that KL-divergence is the loss objective in our framework, we extrapolate that HGDL converges well in terms of minimizing the distribution distance. Same observation can

be drawn from the validation loss curve as shown in Supplement C Figures 5, 6, and 7. In addition, on metrics being strongly related to KL divergence including COD, CHD, and IND, our `HGDL` also enjoys significant performance improvement over other models. Among the metrics, CLD metrics shows a different patterns in terms of KL divergence, we show in Figure 4 that CLD and KL has a tradeoff region in small probability distribution and therefore caused such difference.

Third, the **ablation study** between `HGDL` and its variants, i.e., $\text{HGDL}_{\neg \text{transformer}}$, $\text{HGDL}_{\neg \text{TH}}$ and $\text{HGDL}_{\text{ED}}$, demonstrate clear benefits of topology homogenization and consistency-aware graph transformer in aggregating meta-paths and nodal features for LDL learning for heterogeneous graphs (more results are deferred to the Section E.1 in Supplement C; there, we observe that $\text{HGDL}_{\text{ed}}$ has no improvement in KL-divergence with different edge drop rates compared to $\text{HGDL}_{\neg \text{transformer}}$, which is the model with 0 edge drop rate). We observe in Table 2 that $\text{HGDL}_{\neg \text{transformer}}$ shows comparable performance on ACM and YELP by tying `HGDL` in five and six metrics, respectively; however, `HGDL` outperforms it in all settings in other three datasets. Likewise, $\text{HGDL}_{\neg \text{TH}}$ ties `HGDL` across all metrics in YELP but is inferior to `HGDL` in all settings in other four datasets. $\text{HGDL}_{\text{ED}}$ ties `HGDL` in six and five settings on YELP and URBAN, respectively, but is outperformed by `HGDL` for all other three datasets in all settings. The robust performance of `HGDL` can be attributed to two aspects. On the one hand, the improved results over those ablation variants suggest that our devised model components for proactive meta-path learning and attention modeling are indispensable. On other other hand, it substantiates the usefulness of our design that lets `HGDL` learn semantic fusion before the embedding learning. This end-to-end learning design provides a larger search space for embedding learning to find optimal solutions, whereas other methods that learn embedding and perform fusion in two independent stages may result in suboptimal node embeddings thus inferior LDL performance.

Fourth, even though the optimal meta-path choice may vary across different metrics and datasets, our `HGDL` that proactively learns to aggregate multiple meta-path graphs leads to the best performance in most cases. Figure 3 illustrates the performance from single meta-path graph and we can observe that our method outperforms the single best path results in all five datasets, with a larger improvement when the meta-path results are close (indicating each meta-path has similar information, *e.g.*, ACM dataset in Figure 3 (b)) and a smaller improvement when one meta-path is significantly inferior to others (*e.g.*, DBLP dataset Figure 3 (c) where $p_1$ outperforms $p_2$ with a large margin). These results validate the tightness of Theorem 1 by demonstrating the optimality of the learned meta-path graph in our `HGDL` method.

### 6.3  Scalability Analysis

Denote the total number of nodes, hidden dimension size, and number of meta-path by $n$, $f$, and $k$, respectively. The number of learnable parameters is $\mathcal{O}(n)$ for graph topology homogenization, because `HGDL` requires learning an adjacency matrix from all meta-path, which involves $knf + k^2 f$ training parameters (*i.e.* $\mathcal{O}(n)$ complexity). Inducing adjacency matrix from features, *i.e.* the 2nd stage, only requires $\mathcal{O}(1)$ number of learnable parameters, same as vanilla GCN. As a result, `HGDL` has $\mathcal{O}(n)$ complexity. The runtime performance is detailed in Appendix H.3.

## 7  Conclusion

This paper explored a novel graph learning setting, namely, heterogeneous graph label distribution learning. Our goal is to predict label distributions of target nodes in a heterogeneous graph, which enables a finer-granular delineation of node properties compared to traditional single- or multi-class node classification. We demonstrated that the topological heterogeneity and inconsistency impose unique challenge for generalizing LDL into networked data, and proposed `HGDL` to overcome them. Specifically, `HGDL` proactively aggregates meta-paths to achieve optimal graph topology homogenization through attention mechanism, followed by a transformer-based approach to ensure topology and feature consistency for learning node label distributions. We analyzed the PAC-Bayes error bound of `HGDL`, and the result suggests the superiority of our design over those models learned from a single meta-path graph. Empirical results on five benchmark datasets validated the tightness of our analysis and substantiate that `HGDL` significantly outperformed its competitors.

## Acknowledgment

This work has been supported in part by the National Science Foundation (NSF) under Grant Nos. IIS-2236578, IIS-2236579, IIS-2302786, IIS-2441449, IOS-2430224, and IOS-2446522.

## Footnotes

[2] www.openstreetmap.org

[3] https://www.kaggle.com/c/pkdd-15-predict-taxi-service-trajectory-i/data

## References

[1] N. L. Houssou, J.-l. Guillaume, and A. Prigent, "A graph based approach for functional urban areas delineation," in *Proceedings of the 34th ACM/SIGAPP Symposium on Applied Computing*, pp. 652–658, 2019.

[2] A. Rossi, G. Barlacchi, M. Bianchini, and B. Lepri, "Modelling taxi drivers' behaviour for the next destination prediction," *IEEE Transactions on Intelligent Transportation Systems*, vol. 21, no. 7, pp. 2980–2989, 2019.

[3] P. Kar and P. Jain, "Supervised learning with similarity functions," *NeurIPS*, vol. 25, 2012.

[4] A. Hefny, C. Downey, and G. J. Gordon, "Supervised learning for dynamical system learning," *NeurIPS*, vol. 28, 2015.

[5] X. Geng, "Label distribution learning," *IEEE Trans. Knowl. Data Eng.*, vol. 28, no. 7, pp. 1734–1748, 2016.

[6] X. Jia, W. Li, J. Liu, *et al.*, "Label distribution learning by exploiting label correlations," in *AAAI*, 2018.

[7] T. Ren, X. Jia, W. Li, and S. Zhao, "Label distribution learning with label correlations via low-rank approximation," in *IJCAI*, p. 3325–3331, 2019.

[8] J. Wang and X. Geng, "Theoretical analysis of label distribution learning," in *AAAI*, vol. 33, pp. 5256–5263, 2019.

[9] X. Zhao, L. Qi, Y. An, and X. Geng, "Generalizable label distribution learning," in *Proceedings of the 31st ACM International Conference on Multimedia (MM-23)*, p. 8932–8941, 2023.

[10] T. Ren, X. Jia, W. Li, L. Chen, and Z. Li, "Label distribution learning with label-specific features," in *Proceedings of the Twenty-Eighth International Joint Conference on Artificial Intelligence (IJCAI-19)*, pp. 3318–3324, International Joint Conferences on Artificial Intelligence Organization, 2019.

[11] H. Xu, X. Liu, Q. Zhao, Y. Ma, C. Yan, and F. Dai, "Gaussian label distribution learning for spherical image object detection," in *CVPR*, 2023.

[12] Z. Yao, Y. Fu, B. Liu, W. Hu, and H. Xiong, "Representing urban functions through zone embedding with human mobility patterns," in *IJCAI*, 2018.

[13] Y. Luo, F.-l. Chung, and K. Chen, "Urban region profiling via multi-graph representation learning," in *CIKM*, pp. 4294–4298, 2022.

[14] Y. Zheng, Y. Lin, L. Zhao, T. Wu, D. Jin, and Y. Li, "Spatial planning of urban communities via deep reinforcement learning," *Nature Computational Science*, vol. 3, no. 9, pp. 748–762, 2023.

[15] Y. Liu, J. Ding, Y. Fu, and Y. Li, "Urbankg: An urban knowledge graph system," *ACM Transactions on Intelligent Systems and Technology*, vol. 14, no. 4, pp. 1–25, 2023.

[16] Q. Lv, M. Ding, Q. Liu, Y. Chen, W. Feng, S. He, C. Zhou, J. Jiang, Y. Dong, and J. Tang, "Are we really making much progress? revisiting, benchmarking and refining heterogeneous graph neural networks," in *SIGKDD*, pp. 1150–1160, 2021.

[17] C. Zhang, D. Song, C. Huang, A. Swami, and N. Chawla, "Heterogeneous graph neural network," in *Proc. of KDD*, pp. 793–803, 2019.

[18] X. Fu, J. Zhang, Z. Meng, and I. King, "Magnn: Metapath aggregated graph neural network for heterogeneous graph embedding," in *WWW*, pp. 2331–2341, 2020.

[19] H. Borchani, G. Varando, C. Bielza, and P. Larranaga, "A survey on multi-output regression," *Wiley Interdisciplinary Reviews: Data Mining and Knowledge Discovery*, vol. 5, 07 2015.

[20] Y. Wang, Y. Zhou, J. Zhu, X. Liu, W. Yan, and Z. Tian, "Contrastive label enhancement," in *Proceedings of the Thirty-Second International Joint Conference on Artificial Intelligence (IJCAI-23)*, pp. 4353–4361, 2023.

[21] Y. Jin, R. Gao, Y. He, and X. Zhu, "Gldl: Graph label distribution learning," in *Proceedings of the 38th Annual AAAI Conference on Artificial Intelligence*, 2024.

[22] Y. Dong, Z. Hu, K. Wang, Y. Sun, and J. Tang, "Heterogeneous network representation learning," in *Proc. of the 29th International Joint Conf. on Artificial Intelligence (IJCAI-20)*, pp. 4861–4867, 7 2020.

[23] Y. Jing, Y. Yang, X. Wang, M. Song, and D. Tao, "Amalgamating knowledge from heterogeneous graph neural networks," in *CVPR*, pp. 15709–15718, 2021.

[24] X. Wang, N. Liu, H. Han, and C. Shi, "Self-supervised heterogeneous graph neural network with co-contrastive learning," in *SIGKDD*, pp. 1726–1736, 2021.

[25] W. Xiao, J. Houye, S. Chuan, W. Bai, C. Peng, Y. P., and Y. Yanfang, "Heterogeneous graph attention network," *WWW*, 2019.

[26] Z. Hu, Y. Dong, K. Wang, and Y. Sun, "Heterogeneous graph transformer," in *Proceedings of The Web Conference*, p. 2704–2710, 2020.

[27] X. Yang, M. Yan, S. Pan, X. Ye, and D. Fan, "Simple and efficient heterogeneous graph neural network," *Proceedings of the AAAI Conference on Artificial Intelligence*, pp. 10816–10824, 2023.

[28] M. Wan, Y. Ouyang, L. Kaplan, and J. Han, "Graph regularized meta-path based transductive regression in heterogeneous information network," in *SDM*, pp. 918–926, 2015.

[29] P. Veličković, G. Cucurull, A. Casanova, A. Romero, P. Liò, and Y. Bengio, "Graph attention networks," in *ICLR*, 2018.

[30] V. P. Dwivedi and X. Bresson, "A generalization of transformer networks to graphs," 2021.

[31] Y. Rong, W. Huang, T. Xu, and J. Huang, "Dropedge: Towards deep graph convolutional networks on node classification," in *International Conference on Learning Representations (ICLR)*, 2020.

[32] S. Kullback and R. A. Leibler, "On information and sufficiency," *The annals of mathematical statistics*, vol. 22, no. 1, pp. 79–86, 1951.

[33] J. Tang, J. Zhang, L. Yao, J. Li, L. Zhang, and Z. Su, "Arnetminer: Extraction and mining of academic social networks," in *KDD*, pp. 990–998, 2008.

[34] Y. Gu, S. Zheng, Q. Yin, R. Jiang, and J. Li, "REDDA: Integrating multiple biological relations to heterogeneous graph neural network for drug-disease association prediction," *Computers in Biology and Medicine*, 11 2022.

[35] T. N. Kipf and M. Welling, "Semi-supervised classification with graph convolutional networks," in *ICLR*, 2017.

[36] G. Landrum, P. Tosco, B. Kelley, Ric, D. Cosgrove, sriniker, gedeck, R. Vianello, NadineSchneider, E. Kawashima, G. Jones, D, N, A. Dalke, B. Cole, M. Swain, S. Turk, AlexanderSavelyev, A. Vaucher, M. Wójcikowski, I. Take, V. F. Scalfani, D. Probst, K. Ujihara, guillaume godin, A. Pahl, R. Walker, J. Lehtivarjo, F. Berenger, jasondbiggs, and strets123, "rdkit/rdkit: 2023_09_4 (q3 2023) release," Jan. 2024.

[37] D. S. Himmelstein, "User-friendly extensions to mesh v1.0," Feb. 2016.

[38] V. D. Blondel, J.-L. Guillaume, R. Lambiotte, and E. Lefebvre, "Fast unfolding of communities in large networks," *Journal of statistical mechanics: theory and experiment*, vol. 2008, no. 10, p. P10008, 2008.

[39] Q. Mao, Z. Liu, C. Liu, and J. Sun, "Hinormer: Representation learning on heterogeneous information networks with graph transformer," 2023.

[40] N. Cesa-Bianchi and G. Lugosi, *Prediction, learning, and games*. Cambridge university press, 2006.

[41] P. L. Bartlett, D. J. Foster, and M. J. Telgarsky, "Spectrally-normalized margin bounds for neural networks," *NeurIPS*, vol. 30, 2017.

[42] B. Neyshabur, S. Bhojanapalli, and N. Srebro, "A pac-bayesian approach to spectrally-normalized margin bounds for neural networks," in *ICLR*, 2018.

[43] R. Liao, R. Urtasun, and R. Zemel, "A pac-bayesian approach to generalization bounds for graph neural networks," in *ICLR*, 2020.

[44] J. A. Tropp, "User-friendly tail bounds for sums of random matrices," *Foundations of computational mathematics*, vol. 12, pp. 389–434, 2012.

[45] P. L. Bartlett and S. Mendelson, "Rademacher and gaussian complexities: Risk bounds and structural results," *Journal of Machine Learning Research*, vol. 3, no. Nov, pp. 463–482, 2002.

[46] R. E. Schapire and Y. Freund, *Foundations of machine learning*, pp. 23–52. Mit Press, 2012.

[47] A. Maurer, "A vector-contraction inequality for rademacher complexities," in *ALT*, pp. 3–17, Springer, 2016.

[48] S. M. Kakade, K. Sridharan, and A. Tewari, "On the complexity of linear prediction: Risk bounds, margin bounds, and regularization," *NeurIPS*, vol. 21, 2008.

[49] S.-H. Cha, "Comprehensive survey on distance/similarity measures between probability density functions," *City*, vol. 1, no. 2, p. 1, 2007.

**Algorithm 1** HGDL Algorithm
***

**Input**: $G = \{V, E, X, Y\}$; Meta-paths: $\mathbb{P} = \{\mathcal{P}_1, \ldots, \mathcal{P}_k\}$
**Parameter**: epochs, $U[1, k]$ is the uniform distribution, $\gamma$ is a constant
**Output**: $\hat{Y}$

 1: $\mathcal{A} \leftarrow \emptyset$
 2: **For** <each meta-path $\mathcal{P}_i \in \mathbb{P}$>
 3:      Meta-path Graph $G_i \leftarrow$ Generate Meta-path graph $(G, \mathcal{P}_i)$
 4:      $A_i \leftarrow$ Adjacency Matrix $(G_i)$
 5:      $\mathcal{A} \leftarrow \mathcal{A} \cup A_i$
 6: **EndFor**
 7: $t \leftarrow 0.$
 8: **while** $t \leq$ epochs **do**
 9:      $\Theta \leftarrow \text{Attention}_{W_0^i}(A_i)$ {Attention() follows Eq. (1)}
10:      $\tilde{A} \leftarrow \text{Aggregate}_{W_0'}(\Theta, A_i)$ {Aggregate follows Eq. (2)}
11:      $Z \leftarrow \text{ReLU}(\tilde{A}XW_1)$
12:      $Z_q \leftarrow ZQ$
13:      $Z_k \leftarrow (ZK)^{\top}$
14:      $\tilde{A}_2 \leftarrow \text{LapNorm}\Big(\text{softmax}((Z_q Z_k) \odot \tilde{A})\Big)$
15:      $H \leftarrow \text{LeakyReLU}(\tilde{A}_2 Z W_2)$
16:      $\hat{Y} \leftarrow \text{softmax}(\tilde{A}HW_3)$
17:      $\ell_{KL} \leftarrow D_{\text{KL}}(Y||\hat{Y})$
18:      $\ell_{reg} \leftarrow D_{\text{KL}}(\Theta||U[1, k])$
19:      Gradient $\leftarrow \arg\min(\ell_{KL} - \gamma \cdot \ell_{reg})$
20:      Update $W_1, W_2, W_3, Q, K, W_0^i, W_i'$ with Gradient.
21:      $t \leftarrow t + 1$
22: **end while**
23: **return** $\hat{Y}$
***

## A  Appendix / supplemental material

## B  Roadmap

A structured guide to navigate the supplement content is organized as follows.

- **Supplement A:** Supplement A presents the pseudo-code of our proposed HGDL algorithm.

- **Supplement B:** Supplement B reports datasets and baseline methods used for the experimental study.

- **Supplement C:** Supplement C reports experimental settings and additional results.

- **Supplement D:** Supplement D elaborates the proof of Theorem 1 as introduced in Section 5 of the main manuscript.

- **Supplement E:** Supplement E elaborates the proof of Theorem 2 as introduced in Section 5 of the main manuscript.

- **Supplement F:** Supplement F carries out additional theoretical analysis explaining the rationality of HGDL for label distribution learning.

## C  Supplement A: Pseudo-code and Implementation Details

Algorithm 1 lists the main steps of the HGDL framework where Lines 1 to 6 generate meta-path graphs and their adjacency matrices. Lines 7 to 10 denote the optimal graph topology homogenization (Sec 4.1 of the main manuscript), and Lines 11 to 15 are local topology and global feature consistency-aware graph transformer (Sec 4.2 of the main manuscript). Lines 16 to 20 denote the loss terms and the model training process.

| Dataset | Node Types | # features | Edge Types | # Labels |
|---|---|---|---|---|
| DRUG | D:Drug(894)/S:Disease(454) P:Protein(18877)/G:gene(20561) | 191 | DP(4397)/DS(2704) PG(18545)/PP(201382) GG(712546)/DD(798316) | 28 |
| ACM | A:Author(5810)/P:Paper(12499) S:Subject(73)/C:conference(14) F:Affiliation(1804) | 1903 | AP(37055)/PC(12499) PS(12499)/PF(12499) AF(30424) | 14 |
| DBLP | A:Author(4057)/P:Paper(14328) T:Term(8920)/C:conference(20) | 8920 | AP(19645)/PT(114273) PC(14328) | 4 |
| YELP | U:User(3001)/B:Business(150346) R:Review(6990280)/T:Tip(908915) | 19 | UR(3001)/RB(6990280) UT(3001)/BT(908915) | 9 |
| URBAN | NR:Nature Residence(622) CS:Comprehensive Service(449) GL:Green Leisure(202) TJ:Transit Junction(161) | 155 | NR-CS (14818) NR-GL (8328) NR-TJ (11736) | 10 |

Table 3: Benchmark dataset data statistics

| Datasets | Meta-Paths | Label semantics |
|---|---|---|
| DRUG | dpd, dsd, dpgpd | See Table 6 |
| ACM | apa, afa, apspa | {KDD, SIGMOD, WWW, SIGIR, CIKM, SODA, STOC, SOSP, SPAA, SIGCOMM, MobiCOMM, ICML, COLT, VLDB} |
| DBLP | apcpa, aptpa | {Database, Data Mining, AI, Information Retrieval} |
| YELP | urbru, utbtu | {1, 1.5, 2, 2.5, 3, 3.5, 4, 4.5, 5} |
| URBAN | rsr, rlr, rtr | {Accommodation and Residence, Workplace and Public Service, Sustenance and Leisure Places, Farmland, Transportation, Business and Industry, Education, Healthcare, Greenfield, Worship} |

Table 4: Meta-paths and label semantics of the benchmark datasets

# D    Supplement B: Experiments and Datasets

## D.1    Dataset Description

Because no benchmark heterogeneous graph datasets with label distributions currently exist, we create four datasets using existing heterogeneous graphs including DBLP [33], ACM [33], YELP, and DRUG dataset [34]. Table 3 reports number of nodes/edges, number of each node/edge types, and number of labels for each dataset. Table 4 reports meta-paths and label semantic for each datasets. Number of features are refer to the content attribute/features of the target node types. In our experiments, we did not use any content information of non-target nodes (so only target nodes have features).

- **DBLP** is a citation network [33] with four types of nodes (author, paper, conference, and term nodes). Author nodes are our target nodes. To construct a label distribution, we labeled each author with a distribution of conference areas, using the history of each author's published papers at different conference areas. The author's node features are a bag of words of all papers published by the author.

- **ACM** is a citation network from [33] with five types of nodes (author, paper, conference, affiliation, and subject nodes). Author nodes are our target nodes. We first extract one third of the authors and then labeled them based on the subjects of their published papers. Similar to the DBLP dataset, each author's features are also a bag-of-word representation of the author's papers.

- **YELP** is an open-access review network with four types of nodes (business, user, tip, and review nodes). User nodes are our target nodes. We label each user by counting the ratings of the business each user made reviews on and normalize the countings of each rate to a distribution. User's features are the bag of words of their reviews.

| Drug name | Drug ID | Drug SMILES |
|---|---|---|
| Clobazam | DB00734 | Cc1nc2n(c(=O)c1CCN1CCC(c3noc4cc(F)ccc34)CC1)CCCC2 |
| Vinorelbine | DB00987 | Nc1ccn([CH]2O[CH](CO)[CH](O)[CH]2O)c(=O)n1 |
| Carvedilol | DB00812 | CCCCC1C(=O)N(c2ccccc2)N(c2ccccc2)C1=O |

Table 5: Examples of drug name, ID, and drug in SMILES notation (SMILES: Simplified Molecular Input Line Entry System)

| DRUG prefix MeSH code | Name |
|---|---|
| C01 | bacterial infections and mycoses |
| C02 | virus diseases |
| C03 | parasitic diseases |
| C04 | neoplasms |
| C05 | musculoskeletal diseases |
| C06 | digestive system diseases |
| C07 | stomatognathic diseases |
| C08 | respiratory tract diseases |
| C09 | otorhinolaryngologic diseases |
| C10 | nervous system diseases |
| C11 | eye diseases |
| C12 | urologic and male genital diseases |
| C13 | female genital diseases and pregnancy complications |
| C14 | cardiovascular diseases |
| C15 | hemic and lymphatic diseases |
| C16 | congenital, hereditary, and neonatal diseases and abnormalities |
| C17 | skin and connective tissue diseases |
| C18 | nutritional and metabolic diseases |
| C19 | endocrine system diseases |
| C20 | immune system diseases |
| C22 | animal diseases |
| C23 | pathological conditions, signs and symptoms |
| C25 | Chemically-Induced Disorders |
| C26 | Wounds and Injuries |
| E01 | Diagnosis |
| F01 | Behavior and Behavior Mechanisms |
| F03 | Mental Disorders |
| G07 | Physiological Phenomena |

Table 6: DRUG label semantics

| Class 1 | Class 2 | Class 3 | Class 4 |
|---|---|---|---|
| $0.8774 \pm 0.1755$ | $0.0415 \pm 0.1086$ | $0.0131 \pm 0.0619$ | $0.0680 \pm 0.1226$ |
| $0.0628 \pm 0.1188$ | $0.7733 \pm 0.2157$ | $0.0767 \pm 0.1463$ | $0.0873 \pm 0.1536$ |
| $0.0161 \pm 0.0576$ | $0.0438 \pm 0.1023$ | $0.8667 \pm 0.1881$ | $0.0734 \pm 0.1472$ |
| $0.0309 \pm 0.0879$ | $0.0364 \pm 0.0907$ | $0.0266 \pm 0.0757$ | $0.9061 \pm 0.1652$ |

Table 7: DBLP dataset average label distributions (mean±Std) for all nodes *w.r.t.* different classes. To generate this average, label distributions are grouped by their dominant class. Average label distributions are then calculated within their respective groups. Standard deviations are calculated between each class within each individual group. The table is $q \times q$, where $q$ denotes the number of classes. The diagonal values denote the dominant class's probability value. The lower the diagonal values, the more spread out the class probability is.

- **DRUG** is a drug-disease-protein-gene network from [34], with four types of nodes (drug, disease, protein, and gene nodes). Drug nodes are our target nodes. The label distribution of each drug node is obtained by using associated disease etiology (using top-level categories from Medical Subject Headings, MeSH, disease database). The node features is

| Class 1 | Class 2 | Class 3 | Class 4 | Class 5 | Class 6 | Class 7 | Class 8 | Class 9 | Class 10 | Class 11 | Class 12 | Class 13 | Class 14 |
|---|---|---|---|---|---|---|---|---|---|---|---|---|---|
| 0.8363 ± 0.2439 | 0.0049 ± 0.0328 | 0.0045 ± 0.0325 | 0.0067 ± 0.0389 | 0.0112 ± 0.0516 | 0.0012 ± 0.0168 | 0 | 0.0009 ± 0.0152 | 0.0005 ± 0.0084 | 0.0011 ± 0.0154 | 0.0024 ± 0.0255 | 0.0120 ± 0.0548 | 0.0034 ± 0.0274 | 0.0052 ± 0.0349 |
| 0.0204 ± 0.0838 | 0.8317 ± 0.2357 | 0.0059 ± 0.0381 | 0.0017 ± 0.0177 | 0.0124 ± 0.0570 | 0.0006 ± 0.0086 | 0 | 0.0096 ± 0.0496 | 0.0012 ± 0.0196 | 0.0022 ± 0.0265 | 0.0007 ± 0.0101 | 0.0032 ± 0.0306 | 0.0025 ± 0.0254 | 0.0584 ± 0.1232 |
| 0.0317 ± 0.1051 | 0.0234 ± 0.0844 | 0.8915 ± 0.2095 | 0.0142 ± 0.0561 | 0.0172 ± 0.0657 | 0.0047 ± 0.0301 | 0 | 0.0013 ± 0.0149 | 0.0000 ± 0.0000 | 0.0053 ± 0.0362 | 0.0029 ± 0.0292 | 0.0038 ± 0.0304 | 0.0021 ± 0.0192 | 0.0073 ± 0.0379 |
| 0.0203 ± 0.0871 | 0.0054 ± 0.0422 | 0.0270 ± 0.1023 | 0.8915 ± 0.1929 | 0.0245 ± 0.0813 | 0.0006 ± 0.0111 | 0 | 0.0009 ± 0.0152 | 0.0009 ± 0.0147 | 0.0002 ± 0.0045 | 0.0000 ± 0.0000 | 0.0075 ± 0.0468 | 0.0001 ± 0.0022 | 0.0015 ± 0.0194 |
| 0.0417 ± 0.1174 | 0.0404 ± 0.1140 | 0.0374 ± 0.1154 | 0.0707 ± 0.1430 | 0.8913 ± 0.1963 | 0.0028 ± 0.0234 | 0 | 0.0000 ± 0.0000 | 0.0009 ± 0.0147 | 0.0004 ± 0.0091 | 0.0000 ± 0.0000 | 0.0085 ± 0.0490 | 0.0017 ± 0.0163 | 0.0196 ± 0.0682 |
| 0.0029 ± 0.0318 | 0.0010 ± 0.0124 | 0.0022 ± 0.0261 | 0.0008 ± 0.0199 | 0.0014 ± 0.0240 | 0.8567 ± 0.2211 | 0 | 0.0000 ± 0.0000 | 0.0132 ± 0.0553 | 0.0021 ± 0.0190 | 0.0000 ± 0.0000 | 0.0036 ± 0.0347 | 0.0362 ± 0.0956 | 0.0015 ± 0.0221 |
| 0.0000 ± 0.0000 | 0.0000 ± 0.0000 | 0.0000 ± 0.0000 | 0.0000 ± 0.0000 | 0.0000 ± 0.0000 | 0.0000 ± 0.0000 | 0 | 0.9417 ± 0.1556 | 0.0011 ± 0.0139 | 0.0080 ± 0.0449 | 0.0029 ± 0.0292 | 0.0000 ± 0.0000 | 0.0013 ± 0.0142 | 0.0000 ± 0.0000 |
| 0.0022 ± 0.0303 | 0.0051 ± 0.0408 | 0.0028 ± 0.0322 | 0.0021 ± 0.0241 | 0.0020 ± 0.0286 | 0.0326 ± 0.1039 | 0 | 0.0018 ± 0.0304 | 0.9398 ± 0.1521 | 0.0029 ± 0.0273 | 0.0012 ± 0.0176 | 0.0000 ± 0.0000 | 0.0086 ± 0.0436 | 0.0007 ± 0.0111 |
| 0.0023 ± 0.0257 | 0.0050 ± 0.0450 | 0.0035 ± 0.0350 | 0.0034 ± 0.0377 | 0.0028 ± 0.0364 | 0.0052 ± 0.0451 | 0 | 0.0288 ± 0.1063 | 0.0035 ± 0.0344 | 0.9483 ± 0.1359 | 0.0230 ± 0.0870 | 0.0000 ± 0.0000 | 0.0013 ± 0.0148 | 0.0015 ± 0.0221 |
| 0.0023 ± 0.0291 | 0.0011 ± 0.0168 | 0.0044 ± 0.0380 | 0.0015 ± 0.0241 | 0.0026 ± 0.0361 | 0.0016 ± 0.0176 | 0 | 0.0077 ± 0.0550 | 0.0034 ± 0.0343 | 0.0221 ± 0.0841 | 0.9657 ± 0.1108 | 0.0000 ± 0.0000 | 0.0011 ± 0.0151 | 0.0004 ± 0.0060 |
| 0.0186 ± 0.0819 | 0.0012 ± 0.0170 | 0.0028 ± 0.0324 | 0.0042 ± 0.0357 | 0.0047 ± 0.0430 | 0.0049 ± 0.0455 | 0 | 0.0000 ± 0.0000 | 0.0000 ± 0.0000 | 0.0000 ± 0.0000 | 0.0000 ± 0.0000 | 0.9515 ± 0.1258 | 0.0041 ± 0.0339 | 0.0000 ± 0.0000 |
| 0.0056 ± 0.0412 | 0.0017 ± 0.0180 | 0.0040 ± 0.0360 | 0.0013 ± 0.0193 | 0.0048 ± 0.0455 | 0.0879 ± 0.1680 | 0 | 0.0031 ± 0.0364 | 0.0338 ± 0.1139 | 0.0055 ± 0.0451 | 0.0000 ± 0.0000 | 0.0000 ± 0.0000 | 0.9355 ± 0.1413 | 0.0015 ± 0.0221 |
| 0.0135 ± 0.0673 | 0.0754 ± 0.1491 | 0.0101 ± 0.0622 | 0.0019 ± 0.0225 | 0.0249 ± 0.0998 | 0.0008 ± 0.0105 | 0 | 0.0043 ± 0.0354 | 0.0017 ± 0.0295 | 0.0018 ± 0.0205 | 0.0012 ± 0.0176 | 0.0018 ± 0.0246 | 0.0022 ± 0.0265 | 0.9024 ± 0.1813 |

Table 8: ACM dataset average label distributions (mean±Std) for all nodes *w.r.t.* different classes. To generate this average, label distributions are grouped by their dominant class. Average label distributions are then calculated within their respective groups. Standard deviations are calculated between each class within each individual group. The table is $q \times q$, where $q$ denotes the number of classes. The diagonal values denote the dominant class's probability value. The lower the diagonal values, the more spread out the class probability is. Class 7 has 0 label distribution because no samples have Class 7 as the dominate class.

| Class 1 | Class 2 | Class 3 | Class 4 | Class 5 | Class 6 | Class 7 | Class 8 | Class 9 | Class 10 | Class 11 | Class 12 | Class 13 | Class 14 |
|---|---|---|---|---|---|---|---|---|---|---|---|---|---|
| 0.4544 ± 0.3787 | 0.0252 ± 0.0056 | 0.0252 ± 0.0106 | 0.0169 ± 0.0128 | 0.0193 ± 0.0111 | 0.0188 ± 0.0206 | 0.0253 ± 0.0062 | 0.0155 ± 0.0128 | 0.0299 ± 0.0031 | 0.0164 ± 0.0130 | 0.0483 ± 0.0373 | 0.0198 ± 0.0051 | 0.0215 ± 0.0109 | 0.0238 ± 0.0119 |
| 0.0191 ± 0.0140 | 0.2676 ± 0.1421 | 0.0252 ± 0.0110 | 0.0175 ± 0.0141 | 0.0193 ± 0.0111 | 0.0155 ± 0.0125 | 0.0253 ± 0.0062 | 0.0144 ± 0.0103 | 0.0299 ± 0.0031 | 0.0164 ± 0.0130 | 0.0300 ± 0.0064 | 0.0198 ± 0.0051 | 0.0215 ± 0.0109 | 0.0238 ± 0.0119 |
| 0.0238 ± 0.0304 | 0.0252 ± 0.0056 | 0.2823 ± 0.2983 | 0.0169 ± 0.0128 | 0.0193 ± 0.0111 | 0.0155 ± 0.0125 | 0.0253 ± 0.0062 | 0.0144 ± 0.0103 | 0.0299 ± 0.0031 | 0.0164 ± 0.0130 | 0.0300 ± 0.0064 | 0.0198 ± 0.0051 | 0.0215 ± 0.0109 | 0.0238 ± 0.0119 |
| 0.0191 ± 0.0140 | 0.0272 ± 0.0096 | 0.0252 ± 0.0110 | 0.3947 ± 0.3474 | 0.0242 ± 0.0219 | 0.0305 ± 0.0459 | 0.0325 ± 0.0149 | 0.0159 ± 0.0149 | 0.0299 ± 0.0031 | 0.0188 ± 0.0173 | 0.0300 ± 0.0064 | 0.0237 ± 0.0161 | 0.0888 ± 0.0771 | 0.0238 ± 0.0119 |
| 0.0191 ± 0.0140 | 0.0252 ± 0.0056 | 0.0252 ± 0.0110 | 0.0185 ± 0.0160 | 0.3955 ± 0.3112 | 0.0214 ± 0.0260 | 0.0253 ± 0.0062 | 0.0144 ± 0.0103 | 0.0299 ± 0.0031 | 0.0182 ± 0.0175 | 0.0300 ± 0.0064 | 0.0207 ± 0.0082 | 0.0252 ± 0.0079 | 0.0238 ± 0.0119 |
| 0.0255 ± 0.0244 | 0.0413 ± 0.0238 | 0.0292 ± 0.0141 | 0.0222 ± 0.0204 | 0.0194 ± 0.0109 | 0.5248 ± 0.3333 | 0.0344 ± 0.0219 | 0.0144 ± 0.0102 | 0.0299 ± 0.0031 | 0.0165 ± 0.0129 | 0.0322 ± 0.0056 | 0.0204 ± 0.0069 | 0.0275 ± 0.0201 | 0.0255 ± 0.0140 |
| 0.0204 ± 0.0146 | 0.0252 ± 0.0056 | 0.0252 ± 0.0110 | 0.0169 ± 0.0128 | 0.0207 ± 0.0137 | 0.0155 ± 0.0125 | 0.1999 ± 0.2965 | 0.0144 ± 0.0103 | 0.0299 ± 0.0031 | 0.0164 ± 0.0130 | 0.0300 ± 0.0064 | 0.0198 ± 0.0051 | 0.0215 ± 0.0109 | 0.0238 ± 0.0119 |
| 0.0199 ± 0.0146 | 0.0302 ± 0.0188 | 0.0252 ± 0.0110 | 0.0169 ± 0.0128 | 0.0209 ± 0.0148 | 0.0274 ± 0.0557 | 0.0253 ± 0.0062 | 0.4674 ± 0.2631 | 0.0429 ± 0.0199 | 0.0166 ± 0.0130 | 0.0322 ± 0.0056 | 0.0208 ± 0.0081 | 0.0215 ± 0.0109 | 0.0238 ± 0.0119 |
| 0.0191 ± 0.0139 | 0.0252 ± 0.0056 | 0.0252 ± 0.0110 | 0.0169 ± 0.0128 | 0.0193 ± 0.0111 | 0.0201 ± 0.0137 | 0.0253 ± 0.0062 | 0.0229 ± 0.0144 | 0.1166 ± 0.0541 | 0.0477 ± 0.0461 | 0.0238 ± 0.0093 | 0.0215 ± 0.0109 | 0.0239 ± 0.0119 | 0.0299 ± 0.0284 |
| 0.0191 ± 0.0140 | 0.0274 ± 0.0113 | 0.0252 ± 0.0110 | 0.0376 ± 0.0560 | 0.0193 ± 0.0258 | 0.0175 ± 0.0155 | 0.0477 ± 0.0461 | 0.0144 ± 0.0103 | 0.0369 ± 0.0183 | 0.4892 ± 0.3606 | 0.0300 ± 0.0064 | 0.0242 ± 0.0095 | 0.0215 ± 0.0109 | 0.0299 ± 0.0284 |
| 0.0253 ± 0.0346 | 0.0252 ± 0.0056 | 0.0252 ± 0.0110 | 0.0169 ± 0.0128 | 0.0214 ± 0.0141 | 0.0164 ± 0.0143 | 0.0253 ± 0.0062 | 0.0145 ± 0.0002 | 0.0299 ± 0.0031 | 0.0176 ± 0.0125 | 0.1314 ± 0.1014 | 0.0248 ± 0.0143 | 0.0215 ± 0.0109 | 0.0238 ± 0.0119 |
| 0.0200 ± 0.0140 | 0.0252 ± 0.0056 | 0.0252 ± 0.0110 | 0.0273 ± 0.0348 | 0.0231 ± 0.0297 | 0.0163 ± 0.0125 | 0.0253 ± 0.0062 | 0.0172 ± 0.0220 | 0.0299 ± 0.0031 | 0.0175 ± 0.0155 | 0.0339 ± 0.0142 | 0.1887 ± 0.0792 | 0.0391 ± 0.0179 | 0.0337 ± 0.0251 |
| 0.0202 ± 0.0139 | 0.0272 ± 0.0096 | 0.0262 ± 0.0106 | 0.0283 ± 0.0355 | 0.0196 ± 0.0111 | 0.0164 ± 0.0129 | 0.0325 ± 0.0149 | 0.0172 ± 0.0220 | 0.0299 ± 0.0031 | 0.0175 ± 0.0155 | 0.0339 ± 0.0142 | 0.1654 ± 0.0701 | 0.2576 ± 0.1979 | 0.0337 ± 0.0251 |
| 0.0219 ± 0.0217 | 0.0252 ± 0.0056 | 0.0252 ± 0.0110 | 0.0182 ± 0.0161 | 0.0204 ± 0.0119 | 0.0155 ± 0.0125 | 0.0253 ± 0.0062 | 0.0163 ± 0.0125 | 0.0299 ± 0.0031 | 0.0164 ± 0.0104 | 0.0440 ± 0.0234 | 0.0250 ± 0.0473 | 0.0341 ± 0.0152 | 0.2792 ± 0.2911 |
| 0.0191 ± 0.0140 | 0.0252 ± 0.0056 | 0.0252 ± 0.0110 | 0.0600 ± 0.0933 | 0.0206 ± 0.0118 | 0.0158 ± 0.0124 | 0.0253 ± 0.0062 | 0.0144 ± 0.0103 | 0.0299 ± 0.0031 | 0.0171 ± 0.0142 | 0.0300 ± 0.0064 | 0.0256 ± 0.0206 | 0.0215 ± 0.0109 | 0.0239 ± 0.0119 |
| 0.0191 ± 0.0140 | 0.0252 ± 0.0056 | 0.0252 ± 0.0110 | 0.0216 ± 0.0299 | 0.0405 ± 0.0417 | 0.0156 ± 0.0124 | 0.0571 ± 0.0172 | 0.0212 ± 0.0122 | 0.0299 ± 0.0031 | 0.0240 ± 0.0327 | 0.0423 ± 0.0268 | 0.0557 ± 0.0272 | 0.0378 ± 0.0235 | 0.0330 ± 0.0209 |
| 0.0254 ± 0.0205 | 0.0354 ± 0.0170 | 0.0501 ± 0.0335 | 0.0282 ± 0.0305 | 0.0267 ± 0.0238 | 0.0158 ± 0.0123 | 0.0253 ± 0.0062 | 0.0813 ± 0.0913 | 0.0299 ± 0.0031 | 0.0163 ± 0.0131 | 0.0068 ± 0.0064 | 0.0548 ± 0.0337 | 0.0314 ± 0.0201 | 0.0303 ± 0.0186 |
| 0.0191 ± 0.0140 | 0.0252 ± 0.0056 | 0.0252 ± 0.0110 | 0.0169 ± 0.0128 | 0.0227 ± 0.0259 | 0.0199 ± 0.0217 | 0.0325 ± 0.0149 | 0.0144 ± 0.0103 | 0.0369 ± 0.0183 | 0.0233 ± 0.0394 | 0.0339 ± 0.0142 | 0.0256 ± 0.0206 | 0.0275 ± 0.0201 | 0.0243 ± 0.0113 |
| 0.0191 ± 0.0140 | 0.0252 ± 0.0056 | 0.0305 ± 0.0215 | 0.0264 ± 0.0418 | 0.0206 ± 0.0139 | 0.0179 ± 0.0174 | 0.0521 ± 0.0579 | 0.0144 ± 0.0103 | 0.0299 ± 0.0031 | 0.0181 ± 0.0165 | 0.0341 ± 0.0152 | 0.0224 ± 0.0119 | 0.0424 ± 0.0268 | 0.0238 ± 0.0119 |
| 0.0191 ± 0.0140 | 0.0388 ± 0.0232 | 0.0252 ± 0.0110 | 0.0463 ± 0.0652 | 0.0241 ± 0.0156 | 0.0186 ± 0.0170 | 0.0313 ± 0.0102 | 0.0579 ± 0.0319 | 0.0299 ± 0.0031 | 0.0189 ± 0.0176 | 0.0300 ± 0.0064 | 0.0205 ± 0.0084 | 0.0282 ± 0.0229 | 0.0297 ± 0.0320 |
| 0.0191 ± 0.0140 | 0.0252 ± 0.0056 | 0.0254 ± 0.0106 | 0.0169 ± 0.0128 | 0.0193 ± 0.0111 | 0.0155 ± 0.0125 | 0.0253 ± 0.0062 | 0.0243 ± 0.0230 | 0.0299 ± 0.0031 | 0.0198 ± 0.0226 | 0.0300 ± 0.0064 | 0.0198 ± 0.0051 | 0.0215 ± 0.0109 | 0.0517 ± 0.0447 |
| 0.0191 ± 0.0140 | 0.0252 ± 0.0056 | 0.0252 ± 0.0110 | 0.0173 ± 0.0129 | 0.0193 ± 0.0111 | 0.0164 ± 0.0129 | 0.0253 ± 0.0062 | 0.0144 ± 0.0103 | 0.0299 ± 0.0031 | 0.0164 ± 0.0130 | 0.0300 ± 0.0064 | 0.0198 ± 0.0051 | 0.0215 ± 0.0109 | 0.0238 ± 0.0119 |
| 0.0191 ± 0.0140 | 0.0252 ± 0.0056 | 0.0252 ± 0.0110 | 0.0169 ± 0.0128 | 0.0193 ± 0.0111 | 0.0162 ± 0.0136 | 0.0253 ± 0.0062 | 0.0144 ± 0.0102 | 0.0299 ± 0.0031 | 0.0177 ± 0.0157 | 0.0300 ± 0.0064 | 0.0198 ± 0.0051 | 0.0215 ± 0.0109 | 0.0238 ± 0.0119 |
| 0.0191 ± 0.0140 | 0.0252 ± 0.0056 | 0.0252 ± 0.0110 | 0.0169 ± 0.0128 | 0.0193 ± 0.0111 | 0.0155 ± 0.0125 | 0.0253 ± 0.0062 | 0.0144 ± 0.0103 | 0.0377 ± 0.0217 | 0.0421 ± 0.0480 | 0.0300 ± 0.0064 | 0.0208 ± 0.0079 | 0.0215 ± 0.0109 | 0.0238 ± 0.0119 |
| 0.0191 ± 0.0140 | 0.0252 ± 0.0056 | 0.0252 ± 0.0110 | 0.0169 ± 0.0128 | 0.0193 ± 0.0111 | 0.0155 ± 0.0125 | 0.0253 ± 0.0062 | 0.0144 ± 0.0103 | 0.0299 ± 0.0031 | 0.0164 ± 0.0130 | 0.0300 ± 0.0064 | 0.0198 ± 0.0051 | 0.0215 ± 0.0109 | 0.0238 ± 0.0119 |

Table 9: DRUG dataset average label distributions(part 1) (mean±Std) for all nodes *w.r.t.* different classes. To generate this average, label distributions are grouped by their dominant class. Average label distributions are then calculated within their respective groups. Standard deviations are calculated between each class within each individual group. The table is $q \times q$, where $q$ denotes the number of classes. The diagonal values denote the dominant class's probability value. The lower the diagonal values, the more spread out the class probability is.

| Class 15 | Class 16 | Class 17 | Class 18 | Class 19 | Class 20 | Class 21 | Class 22 | Class 23 | Class 24 | Class 25 | Class 26 | Class 27 | Class 28 |
|---|---|---|---|---|---|---|---|---|---|---|---|---|---|
| 0.0191 ± 0.0123 | 0.0251 ± 0.0084 | 0.0213 ± 0.0378 | 0.0231 ± 0.0121 | 0.0251 ± 0.0076 | 0.0220 ± 0.0145 | 0 | 0.0305 ± 0.0054 | 0.0318 ± 0.0000 | 0.0336 ± 0 | 0 | 0.0330 ± 0.0009 | 0.0287 ± 0.0067 | 0 |
| 0.0191 ± 0.0123 | 0.0251 ± 0.0084 | 0.0127 ± 0.0109 | 0.0231 ± 0.0121 | 0.0251 ± 0.0076 | 0.0202 ± 0.0153 | 0 | 0.0305 ± 0.0054 | 0.0318 ± 0.0000 | 0.0336 ± 0 | 0 | 0.0330 ± 0.0009 | 0.0287 ± 0.0067 | 0 |
| 0.0191 ± 0.0123 | 0.0251 ± 0.0084 | 0.0125 ± 0.0110 | 0.0231 ± 0.0121 | 0.0251 ± 0.0076 | 0.0202 ± 0.0153 | 0 | 0.0305 ± 0.0054 | 0.0318 ± 0.0000 | 0.0336 ± 0 | 0 | 0.0330 ± 0.0009 | 0.0287 ± 0.0067 | 0 |
| 0.0375 ± 0.0328 | 0.0276 ± 0.0102 | 0.0152 ± 0.0146 | 0.0231 ± 0.0121 | 0.0368 ± 0.0325 | 0.0353 ± 0.0384 | 0 | 0.0305 ± 0.0054 | 0.0318 ± 0.0173 | 0.0336 ± 0 | 0 | 0.0330 ± 0.0009 | 0.0287 ± 0.0067 | 0 |
| 0.0191 ± 0.0123 | 0.0362 ± 0.0226 | 0.0159 ± 0.0184 | 0.0270 ± 0.0197 | 0.0251 ± 0.0076 | 0.0244 ± 0.0119 | 0 | 0.0305 ± 0.0054 | 0.0318 ± 0.0000 | 0.0336 ± 0 | 0 | 0.0330 ± 0.0009 | 0.0318 ± 0.0107 | 0 |
| 0.0191 ± 0.0123 | 0.0327 ± 0.0201 | 0.0125 ± 0.0110 | 0.0243 ± 0.0125 | 0.0251 ± 0.0076 | 0.0298 ± 0.0238 | 0 | 0.0305 ± 0.0054 | 0.0318 ± 0.0000 | 0.0336 ± 0 | 0 | 0.0330 ± 0.0009 | 0.0287 ± 0.0067 | 0 |
| 0.0191 ± 0.0123 | 0.0251 ± 0.0084 | 0.0130 ± 0.0117 | 0.0231 ± 0.0121 | 0.0251 ± 0.0076 | 0.0202 ± 0.0153 | 0 | 0.0305 ± 0.0054 | 0.0318 ± 0.0000 | 0.0336 ± 0 | 0 | 0.0330 ± 0.0009 | 0.0287 ± 0.0067 | 0 |
| 0.0223 ± 0.0188 | 0.0327 ± 0.0201 | 0.0207 ± 0.0311 | 0.0231 ± 0.0121 | 0.0251 ± 0.0076 | 0.0237 ± 0.0132 | 0 | 0.0444 ± 0.0186 | 0.0318 ± 0.0000 | 0.0336 ± 0 | 0 | 0.0330 ± 0.0009 | 0.0287 ± 0.0067 | 0 |
| 0.0196 ± 0.0117 | 0.0251 ± 0.0084 | 0.0144 ± 0.0184 | 0.0233 ± 0.0118 | 0.0251 ± 0.0076 | 0.0202 ± 0.0153 | 0 | 0.0305 ± 0.0054 | 0.0318 ± 0.0000 | 0.0336 ± 0 | 0 | 0.0330 ± 0.0009 | 0.0287 ± 0.0067 | 0 |
| 0.0197 ± 0.0115 | 0.0486 ± 0.0413 | 0.0125 ± 0.0109 | 0.0243 ± 0.0117 | 0.0251 ± 0.0076 | 0.0257 ± 0.0113 | 0 | 0.0305 ± 0.0054 | 0.0318 ± 0.0000 | 0.0336 ± 0 | 0 | 0.0330 ± 0.0009 | 0.0314 ± 0.0116 | 0 |
| 0.0193 ± 0.0121 | 0.0251 ± 0.0084 | 0.0125 ± 0.0109 | 0.0231 ± 0.0121 | 0.0364 ± 0.0144 | 0.0202 ± 0.0153 | 0 | 0.0305 ± 0.0054 | 0.0318 ± 0.0000 | 0.0336 ± 0 | 0 | 0.0330 ± 0.0009 | 0.0287 ± 0.0067 | 0 |
| 0.0241 ± 0.0173 | 0.0284 ± 0.0133 | 0.0167 ± 0.0253 | 0.0264 ± 0.0210 | 0.0251 ± 0.0076 | 0.0205 ± 0.0149 | 0 | 0.0305 ± 0.0054 | 0.0318 ± 0.0000 | 0.0336 ± 0 | 0 | 0.0330 ± 0.0009 | 0.0301 ± 0.0079 | 0 |
| 0.0241 ± 0.0173 | 0.0284 ± 0.0133 | 0.0216 ± 0.0326 | 0.0264 ± 0.0210 | 0.0291 ± 0.0164 | 0.0205 ± 0.0149 | 0 | 0.0305 ± 0.0054 | 0.0318 ± 0.0000 | 0.0336 ± 0 | 0 | 0.0330 ± 0.0009 | 0.0301 ± 0.0079 | 0 |
| 0.0302 ± 0.0192 | 0.0251 ± 0.0084 | 0.0128 ± 0.0110 | 0.0257 ± 0.0177 | 0.0251 ± 0.0076 | 0.0220 ± 0.0145 | 0 | 0.0305 ± 0.0054 | 0.0318 ± 0.0000 | 0.0336 ± 0 | 0 | 0.0330 ± 0.0009 | 0.0287 ± 0.0067 | 0 |
| 0.3424 ± 0.3057 | 0.0302 ± 0.0115 | 0.0129 ± 0.0111 | 0.0231 ± 0.0121 | 0.0251 ± 0.0076 | 0.0538 ± 0.0608 | 0 | 0.0305 ± 0.0054 | 0.0318 ± 0.0000 | 0.0336 ± 0 | 0 | 0.0330 ± 0.0009 | 0.0287 ± 0.0067 | 0 |
| 0.0336 ± 0.0305 | 0.2156 ± 0.1776 | 0.0227 ± 0.0284 | 0.0380 ± 0.0345 | 0.0251 ± 0.0076 | 0.0226 ± 0.0135 | 0 | 0.0305 ± 0.0054 | 0.0318 ± 0.0000 | 0.0336 ± 0 | 0 | 0.0330 ± 0.0009 | 0.0287 ± 0.0067 | 0 |
| 0.0227 ± 0.0141 | 0.0251 ± 0.0084 | 0.5931 ± 0.3144 | 0.0237 ± 0.0124 | 0.0251 ± 0.0076 | 0.0557 ± 0.0654 | 0 | 0.0305 ± 0.0054 | 0.0318 ± 0.0000 | 0.0336 ± 0 | 0 | 0.0330 ± 0.0009 | 0.0287 ± 0.0067 | 0 |
| 0.0262 ± 0.0293 | 0.0579 ± 0.0455 | 0.0130 ± 0.0119 | 0.2965 ± 0.3181 | 0.0251 ± 0.0076 | 0.0202 ± 0.0153 | 0 | 0.0444 ± 0.0186 | 0.0318 ± 0.0000 | 0.0336 ± 0 | 0 | 0.0330 ± 0.0009 | 0.0287 ± 0.0067 | 0 |
| 0.0193 ± 0.0121 | 0.0251 ± 0.0084 | 0.0171 ± 0.0236 | 0.0510 ± 0.0434 | 0.2839 ± 0.1802 | 0.0225 ± 0.0149 | 0 | 0.0305 ± 0.0054 | 0.0318 ± 0.0000 | 0.0336 ± 0 | 0 | 0.0330 ± 0.0009 | 0.0287 ± 0.0067 | 0 |
| 0.0830 ± 0.0987 | 0.0251 ± 0.0084 | 0.0236 ± 0.0220 | 0.0266 ± 0.0179 | 0.0364 ± 0.0144 | 0.3397 ± 0.3202 | 0 | 0.0305 ± 0.0054 | 0.0318 ± 0.0000 | 0.0336 ± 0 | 0 | 0.0330 ± 0.0009 | 0.0287 ± 0.0067 | 0 |
| 0.0191 ± 0.0123 | 0.0251 ± 0.0084 | 0.0125 ± 0.0110 | 0.0231 ± 0.0121 | 0.0251 ± 0.0076 | 0.0202 ± 0.0153 | 0 | 0.0305 ± 0.0054 | 0.0318 ± 0.0000 | 0.0336 ± 0 | 0 | 0.0330 ± 0.0009 | 0.0287 ± 0.0067 | 0 |
| 0.0272 ± 0.0275 | 0.0294 ± 0.0185 | 0.0149 ± 0.0176 | 0.0287 ± 0.0207 | 0.0251 ± 0.0076 | 0.0220 ± 0.0145 | 0 | 0.1207 ± 0.0506 | 0.0318 ± 0.0000 | 0.0336 ± 0 | 0 | 0.0330 ± 0.0009 | 0.0313 ± 0.0115 | 0 |
| 0.0191 ± 0.0123 | 0.0251 ± 0.0084 | 0.0132 ± 0.0131 | 0.0231 ± 0.0121 | 0.0251 ± 0.0076 | 0.0220 ± 0.0145 | 0 | 0.0305 ± 0.0054 | 0.0318 ± 0.0000 | 0.0915 ± 0 | 0 | 0.0330 ± 0.0009 | 0.0287 ± 0.0067 | 0 |
| 0.0191 ± 0.0123 | 0.0251 ± 0.0084 | 0.0125 ± 0.0110 | 0.0231 ± 0.0121 | 0.0251 ± 0.0076 | 0.0202 ± 0.0153 | 0 | 0.0305 ± 0.0054 | 0.0318 ± 0.0000 | 0.0336 ± 0 | 0 | 0.0330 ± 0.0009 | 0.0287 ± 0.0067 | 0 |
| 0.0191 ± 0.0123 | 0.0251 ± 0.0084 | 0.0125 ± 0.0110 | 0.0284 ± 0.0209 | 0.0251 ± 0.0076 | 0.0202 ± 0.0153 | 0 | 0.0444 ± 0.0186 | 0.0318 ± 0.0000 | 0.0336 ± 0 | 0 | 0.0330 ± 0.0009 | 0.0287 ± 0.0067 | 0 |
| 0.0193 ± 0.0121 | 0.0251 ± 0.0084 | 0.0125 ± 0.0110 | 0.0231 ± 0.0121 | 0.0251 ± 0.0076 | 0.0202 ± 0.0153 | 0 | 0.0305 ± 0.0054 | 0.0318 ± 0.0000 | 0.0336 ± 0 | 0 | 0.0898 ± 0.0026 | 0.0336 ± 0.0121 | 0 |
| 0.0193 ± 0.0121 | 0.0303 ± 0.0188 | 0.0125 ± 0.0110 | 0.0239 ± 0.0137 | 0.0251 ± 0.0076 | 0.0202 ± 0.0153 | 0 | 0.0305 ± 0.0054 | 0.0865 ± 0.0000 | 0.0336 ± 0 | 0 | 0.0513 ± 0.0273 | 0.2045 ± 0.1652 | 0 |
| 0.0191 ± 0.0123 | 0.0251 ± 0.0084 | 0.0125 ± 0.0110 | 0.0284 ± 0.0209 | 0.0251 ± 0.0076 | 0.0202 ± 0.0153 | 0 | 0.0444 ± 0.0186 | 0.0318 ± 0.0000 | 0.0336 ± 0 | 0 | 0.0330 ± 0.0009 | 0.0287 ± 0.0067 | 0 |

Table 10: DRUG dataset average label distributions(part 2) (mean±Std) for all nodes *w.r.t.* different classes. To generate this average, label distributions are grouped by their dominant class. Average label distributions are then calculated within their respective groups. Standard deviations are calculated between each class within each individual group. The table is $q \times q$, where $q$ denotes the number of classes. The diagonal values denote the dominant class's probability value. The lower the diagonal values, the more spread out the class probability is. Class 21, Class 25, and Class 28 has 0 label distribution because no samples have any one of these three three classes (21, 25, 28) as the dominate class

created from chemical compound of each drug using their SMILES notations. SMILES (Simplified Molecular Input Line Entry System) is a chemical notation that represents a chemical structure as a string so it can be used by the computer. Table 5 lists examples

| Class 1 | Class 2 | Class 3 | Class 4 | Class 5 | Class 6 | Class 7 | Class 8 | Class 9 |
|---|---|---|---|---|---|---|---|---|
| 0.6166 ± 0.2722 | 0.0528 ± 0.0882 | 0.0097 ± 0.0450 | 0.1056 ± 0.1204 | 0.0255 ± 0.0774 | 0.0109 ± 0.0521 | 0.0000 ± 0.0000 | 0.0324 ± 0.0685 | 0.0241 ± 0.0701 |
| 0.0938 ± 0.1400 | 0.6153 ± 0.2831 | 0.0196 ± 0.0634 | 0.1046 ± 0.1170 | 0.0117 ± 0.0485 | 0.0000 ± 0.0000 | 0.0000 ± 0.0000 | 0.0585 ± 0.0976 | 0.0144 ± 0.0515 |
| 0.0290 ± 0.0848 | 0.0409 ± 0.0934 | 0.7113 ± 0.2697 | 0.0235 ± 0.0497 | 0.0021 ± 0.0189 | 0.0109 ± 0.0521 | 0.0000 ± 0.0000 | 0.0368 ± 0.0771 | 0.0074 ± 0.0320 |
| 0.1675 ± 0.1790 | 0.1583 ± 0.1725 | 0.1130 ± 0.1811 | 0.6573 ± 0.2638 | 0.0212 ± 0.0616 | 0.0217 ± 0.0720 | 0.0000 ± 0.0000 | 0.0695 ± 0.1109 | 0.0144 ± 0.0515 |
| 0.0267 ± 0.0816 | 0.0198 ± 0.0755 | 0.0117 ± 0.0629 | 0.0196 ± 0.0633 | 0.8670 ± 0.2324 | 0.0000 ± 0.0000 | 0.0000 ± 0.0000 | 0.0034 ± 0.0205 | 0.0028 ± 0.0167 |
| 0.0053 ± 0.0338 | 0.0095 ± 0.0469 | 0.0172 ± 0.0755 | 0.0071 ± 0.0458 | 0.0076 ± 0.0574 | 0.8261 ± 0.2435 | 0.0000 ± 0.0000 | 0.0059 ± 0.0302 | 0.0028 ± 0.0167 |
| 0.0011 ± 0.0199 | 0.0017 ± 0.0208 | 0.0014 ± 0.0168 | 0.0014 ± 0.0221 | 0.0000 ± 0.0000 | 0.0217 ± 0.1043 | 0.8750 ± 0.2315 | 0.0025 ± 0.0169 | 0.0000 ± 0.0000 |
| 0.0481 ± 0.1050 | 0.0830 ± 0.1300 | 0.0686 ± 0.1510 | 0.0671 ± 0.1061 | 0.0446 ± 0.1255 | 0.0652 ± 0.1722 | 0.0625 ± 0.1768 | 0.7568 ± 0.2872 | 0.0069 ± 0.0417 |
| 0.0118 ± 0.0523 | 0.0186 ± 0.0631 | 0.0475 ± 0.1352 | 0.0139 ± 0.0475 | 0.0203 ± 0.0970 | 0.0435 ± 0.1441 | 0.0625 ± 0.1768 | 0.0343 ± 0.1022 | 0.9273 ± 0.1932 |

Table 11: YELP dataset average label distributions (mean±Std) for all nodes *w.r.t.* different classes. To generate this average, label distributions are grouped by their dominant class. Average label distributions are then calculated within their respective groups. Standard deviations are calculated between each class within each individual group. The table is $q \times q$, where $q$ denotes the number of classes. The diagonal values denote the dominant class's probability value. The lower the diagonal values, the more spread out the class probability is. 0 indicates no samples have specific dominate class.

| Class 1 | Class 2 | Class 3 | Class 4 | Class 5 | Class 6 | Class 7 | Class 8 | Class 9 | Class 10 |
|---|---|---|---|---|---|---|---|---|---|
| 0.6855 ± 0.2143 | 0.0069 ± 0.0475 | 0.0273 ± 0.0792 | 0.0759 ± 0.1119 | 0.0427 ± 0.0949 | 0.0305 ± 0.0699 | 0.0323 ± 0.0884 | 0.0063 ± 0.0449 | 0.0718 ± 0.1116 | 0.0209 ± 0.0564 |
| 0.0064 ± 0.0137 | 0.6357 ± 0.2011 | 0.1042 ± 0.1345 | 0.0234 ± 0.0492 | 0.0522 ± 0.1020 | 0.0321 ± 0.1098 | 0.0388 ± 0.0742 | 0.0358 ± 0.0803 | 0.0471 ± 0.0975 | 0.0244 ± 0.0843 |
| 0.0355 ± 0.0686 | 0.0039 ± 0.0320 | 0.6177 ± 0.1745 | 0.0645 ± 0.1026 | 0.0912 ± 0.1208 | 0.0065 ± 0.0253 | 0.0355 ± 0.0873 | 0.0109 ± 0.0489 | 0.1106 ± 0.1440 | 0.0236 ± 0.0643 |
| 0.0715 ± 0.1133 | 0.0006 ± 0.0068 | 0.0319 ± 0.0893 | 0.6724 ± 0.2079 | 0.0339 ± 0.0869 | 0.0226 ± 0.0765 | 0.0485 ± 0.1025 | 0.0037 ± 0.0285 | 0.0968 ± 0.1271 | 0.0181 ± 0.0581 |
| 0.0655 ± 0.0964 | 0.0249 ± 0.0950 | 0.0470 ± 0.1001 | 0.0588 ± 0.1016 | 0.5907 ± 0.1826 | 0.0200 ± 0.0496 | 0.0581 ± 0.1077 | 0.0296 ± 0.0853 | 0.0837 ± 0.1143 | 0.0216 ± 0.0638 |
| 0.0807 ± 0.1251 | 0.0151 ± 0.0787 | 0.0123 ± 0.0571 | 0.0039 ± 0.0107 | 0.0805 ± 0.1292 | 0.6902 ± 0.2192 | 0.0186 ± 0.0581 | 0.0005 ± 0.0028 | 0.0981 ± 0.1394 | 0.0000 ± 0.0000 |
| 0.0651 ± 0.0921 | 0.0120 ± 0.0552 | 0.0448 ± 0.0999 | 0.0553 ± 0.1193 | 0.0775 ± 0.1190 | 0.0187 ± 0.0587 | 0.6171 ± 0.2053 | 0.0084 ± 0.0363 | 0.0528 ± 0.0863 | 0.0483 ± 0.0986 |
| 0.0198 ± 0.0375 | 0.0092 ± 0.0419 | 0.0227 ± 0.0631 | 0.0445 ± 0.0812 | 0.1065 ± 0.1305 | 0.0034 ± 0.0095 | 0.0344 ± 0.0755 | 0.6798 ± 0.2135 | 0.0597 ± 0.0909 | 0.0201 ± 0.0542 |
| 0.0683 ± 0.1106 | 0.0044 ± 0.0361 | 0.0119 ± 0.0543 | 0.1223 ± 0.1512 | 0.0584 ± 0.1106 | 0.0282 ± 0.0669 | 0.0183 ± 0.0592 | 0.0058 ± 0.0321 | 0.6759 ± 0.2015 | 0.0066 ± 0.0316 |
| 0.0478 ± 0.0898 | 0.0106 ± 0.0573 | 0.0225 ± 0.0564 | 0.1027 ± 0.1356 | 0.0618 ± 0.1078 | 0.0305 ± 0.0705 | 0.0309 ± 0.1285 | 0.0019 ± 0.0103 | 0.0571 ± 0.0908 | 0.5740 ± 0.1959 |

Table 12: URBAN dataset average label distributions (mean±Std) for all nodes *w.r.t.* different classes. To generate this average, label distributions are grouped by their dominant class. Average label distributions are then calculated within their respective groups. Standard deviations are calculated between each class within each individual group. The table is $q \times q$, where $q$ denotes the number of classes. The diagonal values denote the dominant class's probability value. The lower the diagonal values, the more spread out the class probability is. 0 indicates no samples have specific dominate class.

of three drugs' names, IDs, and their SMILES strings. Because SMILES describes drug's chemical compound, We use RDKit library [36] to convert the chemical compound to 191 measures(scalars) of that chemical compound and use it as drug node (target node) features. To obtain the label, we use the MeSH-tree parsing tool [37] to convert the disease ID to corresponding MeSH prefix code which corresponds to high level disease and obtain in total 28 high-level diseases that drugs could potentially target on. Table 6 lists the 28 labels used in DRUG dataset (including MeSH prefix codes and names).

- **URBAN** is a functional urban area network constructed based on the city of Porto, Portugal, uniformly divided into a grid of 30 by 80 blocks. A dataset comprising 22,208 points of interest (POI) is sourced from the OpenStreetMap platform [2] to generate the label distribution. This dataset includes a large amount of fundamental categories like hotels, banks and churches, which have been further categorized into 10 more general groups, as detailed in Table 4. We count the types and corresponding numbers of POIs to generate the label for each block. After excluding blocks devoid of any POIs, the remaining blocks were primarily divided into four different types of nodes using the Louvain algorithm [38]. These node types are named as Nature Residence, Comprehensive Service, Green Leisure and Transit Junction based on their main functions and nodes from Nature Residence are our target nodes. The edges are constructed by Taxi Service Trajectory [3], which documents the taxi routines (from origins to destinations) in Porto. In our resulting heterogeneous graph, two target nodes are considered neighbors if they have the same recorded destination.

Tables 7, 8, 9 and 10, and 11 report average label distributions for DBLP, ACM, DRUG, YELP, and URBAN dataset, respectively. To generate average label distributions, for each dataset, nodes are grouped by their dominant class (where dominant class is the class with the largest distribution value among all classes, using ground-truth distributions). Average label distributions are then calculated within respective groups. It is worth noting that some classes may not be a domain class of any

node (e.g. Class 7 of the ACM dataset in Table 8, or Classes 21, 25, and 28 in the DRUG dataset in Table 10). In this case, the average label distributions of the class are reported as 0 in the tables.

## D.2 Baselines

We extend existing heterogeneous graph learning methods to integrate KL-divergence loss, and implement three baselines for comparisons. In addition, three variants of the proposed `HGDL` method are compared for ablation study purposes:

- $GCN_{KL}$: A baseline uses graph constructed from each meta-path to train a vanilla GCN, using KL-divergence as loss function, and reports the best meta-path result. This baseline is used to demonstrate whether the proposed `HGDL` can outperform the best single meta-path result.

- $HAN_{KL}$: This baseline is to compare `HGDL`'s semantic fusion component with HAN [25], a classical heterogeneous network learning semantic fusion method. $HAN_{KL}$ uses HAN to integrate embedding from meta-path graphs created from different meta-paths, and uses KL-divergence as the loss function for training.

- $SeHGNN_{KL}$ This baseline is to compare `HGDL`'s semantic fusion with SeHGNN [27], a state of the art transformer based heterogeneous semantic fusion learning method. $SeHGNN_{KL}$ uses neighbor aggregation proposed by SeHGNN to aggregate meta-paths embedding, with KL-divergence being used as the loss function for training.

- HGDL: The proposed HGDL method.

- $HGDL_{\neg TH}$: This is a variant of the `HGDL` for ablation study. $HGDL_{\neg TH}$ removes the graph topology homogenization module (Sec 4.1). The embedding is trained from each meta-path with topology and feature consistence-aware graph transformer (Sec 4.2) being applied to each meta-path. The result of the best meta-path is reported, and the purpose is to demonstrate HGDL's topology homogenization effectiveness.

- $HGDL_{\neg transformer}$: This is a variant of the `HGDL` for ablation study. $HGDL_{\neg transformer}$ uses GCN, instead of the transformer (Sec 4.2), to learn embeddings. This serves as an ablation study to show the advantage of HGDL's transformer component for embedding learning.

- $HGDL_{ED}$ This is a variant of the `HGDL` for ablation study. $HGDL_{ED}$ replaces HGDL's topology and feature consistence-aware graph transformer (Sec. 4.2) using random edge dropout method [31]. The purposes to demonstrate the advantage of consistent-aware graph transfer, which resembles to learnable edge dropping, compared to random edge dropout.

## D.3 Experiment Setup

All results reported in the paper are based on 5 times average for each method. Hyperparameters of all the baselines include the hidden dimension for the embedding learning and the hidden dimension for the semantic fusion learning (if the semantic fusion part exists), and the learning rate. In practice, a 0.005 learning rate is fixed for all the baselines unless it is observed that specific method cannot converge under such learning rate or can converge much faster with a large learning rate. An early stop `patience` signal is set and a maximum number of `epochs` is fixed. In practice, we observed convergence before reaching maximum number of epochs in general. For the methods that cannot learn well with all the hyperparameter search, we report the best results we got. For the $HDGL_{ED}$ method, the originally learned feature topology for `HGDL` is replaced by a 0-1 matrix generated by a Bernoulli distribution with a drop rate provided to convert to a 0-1 matrix. The drop rate for the $HDGL_{ED}$ baseline in Table 2 in the main manuscript is 0.1. Fig 5 provides the results of different drop rate cases. For `HGDL` we have additional hyperparameters negative slope of Leaky Relu activation and $\gamma$ for regularization loss.

The train:validation:test splits for DRUG, ACM, DBLP, YELP, and URBAN dataset are 8:1:1, 7:1:2, 4:1:5, 8:1:1, 5:3:2, respectively. We used a smaller training set (40%) for DBLP because because this dataset has comprehensive nodal features (8,920 features) which are the-bag-of-words of papers published by authors. DBLP also has much less number of labels (4 classes), compared to other datasets. The rich nodal features, combined with less number of classes, provide rather rich information to derive research fields. So we reduce the training set size, compared to the test set, to provide less label information for learning.

### D.4 Evaluation metric

Given one predicted label distribution $\hat{y}$ and its corresponding ground truth distribution $y$, the six metrics used in the Table 2 can be defined as the following:

- Cosine Distance(COD):

$$\text{COD}(y, \hat{y}) = 1 - \frac{y \cdot v}{\|y\|\|\hat{y}\|} \tag{4}$$

- Canberra Distance(CAD):

$$\text{CAD}(y, \hat{y}) = \sum_i \frac{|y_i - \hat{y}_i|}{|y_i| + |\hat{y}_i|} \tag{5}$$

- Chebyshev Distance(CHD):

$$\text{CHD}(y, \hat{y}) = \max_i |y_i - \hat{y}_i| \tag{6}$$

- Clark Distance (CLD):

$$\text{CLD}(y, \hat{y}) = \sqrt{\sum_i \frac{(y_i - \hat{y}_i)^2}{(y_i + \hat{y}_i)^2}} \tag{7}$$

- Intersection Score(IND):

$$\text{IND}(y, \hat{y}) = \sum_i \min(y_i, \hat{y}_i) \tag{8}$$

- Kullback-Leibler Divergence(KL):

$$\text{KL}(y, \hat{y}) = \sum_i y_i \cdot \log \frac{y_i}{\hat{y}_i}, \tag{9}$$

## E  Supplement C: Additional Results and Analysis

### E.1  Edge Dropping Comparisons

`HGDL` treats the learnt feature attention as another adjacency matrix, and incorporates it into the graph adjacency matrix by a $\odot$ operator, which provides an intersection of neighbor sets for a node, *i.e.*, the resulting convolution layer is propagated to a neighbor set that both feature and graph topology agrees on. This operation is similar to the recently proposed drop message idea that randomly masks edges for the nodes. Instead of random masking, HGDL selects the neighbors that both feature and graph topology agree on. In other words, Eq. (2) in the main manuscript functions similarly to the edge dropout [31]; in lieu of randomly removing edges, we enforce a neighbor-set intersection, where the information is only propagated from the neighbors on which the feature space and meta-path topology both agree.

To compare the performance between our approach *vs.* the edge dropout [31], the variant HGDL$_{\text{ED}}$ is used to replace our transformer based approach by using random edge dropout. In this case, for HDGL$_{\text{ED}}$, the originally learned feature topology for `HGDL` is replaced by a 0-1 matrix generated by a Bernoulli distribution with a drop rate provided to convert to a 0-1 matrix. For comparisions, we vary the edge drop rates and report HGDL$_{\text{ED}}$'s performance in Figure 5. The results show that that HGDL$_{\text{ED}}$ with random edge dropping doesn't necessarily help improve the distribution learning, implying that our method works not just by relying on the decreasing of the edge number, but also relying on choosing the proper edges to drop.

### E.2  Validation Loss Comparisons

Figures 6, 7, and 8 report the training process validation loss (*w.r.t.* KL-divergence loss) between HAN$_{\text{KL}}$, SeHGNN$_{\text{KL}}$, and `HGDL`. Because early stop is applied to all methods, they terminate at respective number of epochs. In our experiments, a `patience` score is used to determine the minimum loss. A method will continue if a smaller validation loss is found within `patience` number of epochs after the current smallest loss, and terminates otherwise.

| DRUG | | | | | | |
|---|---|---|---|---|---|---|
| Model | COD↓ | CAD↓ | CHD↓ | CLD↓ | IND↑ | KL↓ |
| $\text{GCN}_{\text{KL}}$ | $0.220_{\pm.025}$ | $9.209_{\pm.740}$ | $0.245_{\pm.031}$ | $1.963_{\pm.134}$ | $0.676_{\pm.029}$ | $0.484_{\pm.075}$ |
| $\text{HAN}_{\text{KL}}$ | $0.279_{\pm.023}$ | $10.084_{\pm.823}$ | $0.268_{\pm.027}$ | $2.155_{\pm.148}$ | $0.632_{\pm.025}$ | $0.579_{\pm.055}$ |
| $\text{SeHGNN}_{\text{KL}}$ | $0.286_{\pm.018}$ | $10.178_{\pm.781}$ | $0.267_{\pm.020}$ | $2.166_{\pm.143}$ | $0.640_{\pm.016}$ | $0.600_{\pm.059}$ |
| GLDL | $0.199_{\pm.064}$ | $\mathbf{8.887}_{\pm.838}$ | $0.232_{\pm.044}$ | $\mathbf{1.89}_{\pm.17}$ | $0.699_{\pm.049}$ | $0.409_{\pm.105}$ |
| HINormer | $0.365_{\pm.028}$ | $10.563_{\pm.571}$ | $0.308_{\pm.029}$ | $2.227_{\pm.099}$ | $0.575_{\pm.023}$ | $0.742_{\pm.102}$ |
| HGDL | $\mathbf{0.168}_{\pm.019}$ | $9.179_{\pm.574}$ | $\mathbf{0.217}_{\pm.017}$ | $1.957_{\pm.114}$ | $\mathbf{0.710}_{\pm.020}$ | $\mathbf{0.392}_{\pm.044}$ |
| $\text{HGDL}_{\neg\text{transformer}}$ | $0.199_{\pm.014}$ | $9.371_{\pm.679}$ | $0.235_{\pm.017}$ | $2.004_{\pm.137}$ | $0.687_{\pm.021}$ | $0.492_{\pm.059}$ |
| $\text{HGDL}_{\neg t}$ | $0.212_{\pm.023}$ | $9.510_{\pm.602}$ | $0.240_{\pm.018}$ | $2.029_{\pm.110}$ | $0.671_{\pm.020}$ | $0.462_{\pm.050}$ |
| $\text{HGDL}_{ed}$ | $0.204_{\pm.026}$ | $9.602_{\pm.882}$ | $0.239_{\pm.028}$ | $2.040_{\pm.162}$ | $0.681_{\pm.030}$ | $0.574_{\pm.085}$ |
| DBLP | | | | | | |
| Model | COD↓ | CAD↓ | CHD↓ | CLD↓ | IND↑ | KL↓ |
| $\text{GCN}_{\text{KL}}$ | $0.031_{\pm.004}$ | $2.852_{\pm.009}$ | $0.091_{\pm.006}$ | $1.647_{\pm.002}$ | $0.908_{\pm.006}$ | $0.114_{\pm.011}$ |
| $\text{HAN}_{\text{KL}}$ | $0.025_{\pm.002}$ | $2.819_{\pm.012}$ | $0.071_{\pm.004}$ | $1.633_{\pm.007}$ | $0.929_{\pm.004}$ | $0.082_{\pm.008}$ |
| $\text{SeHGNN}_{\text{KL}}$ | $0.086_{\pm.140}$ | $2.887_{\pm.170}$ | $0.155_{\pm.208}$ | $\mathbf{1.624}_{\pm.049}$ | $0.842_{\pm.214}$ | $0.252_{\pm.397}$ |
| GLDL | $0.019_{\pm.001}$ | $2.8_{\pm.009}$ | $\mathbf{0.056}_{\pm.002}$ | $1.633_{\pm.004}$ | $0.943_{\pm.003}$ | $0.06_{\pm.003}$ |
| HINormer | $0.053_{\pm.008}$ | $2.936_{\pm.013}$ | $0.146_{\pm.019}$ | $1.669_{\pm.008}$ | $0.853_{\pm.019}$ | $0.22_{\pm.025}$ |
| HGDL | $\mathbf{0.019}_{\pm.002}$ | $\mathbf{2.796}_{\pm.014}$ | $0.057_{\pm.005}$ | $1.633_{\pm.005}$ | $\mathbf{0.943}_{\pm.005}$ | $\mathbf{0.057}_{\pm.011}$ |
| $\text{HGDL}_{\neg\text{transformer}}$ | $0.025_{\pm.002}$ | $2.828_{\pm.004}$ | $0.074_{\pm.005}$ | $1.642_{\pm.001}$ | $0.925_{\pm.005}$ | $0.090_{\pm.008}$ |
| $\text{HGDL}_{\neg t}$ | $0.020_{\pm.002}$ | $2.808_{\pm.013}$ | $0.062_{\pm.005}$ | $1.637_{\pm.005}$ | $0.937_{\pm.005}$ | $0.070_{\pm.011}$ |
| $\text{HGDL}_{ed}$ | $0.023_{\pm.001}$ | $2.819_{\pm.005}$ | $0.070_{\pm.003}$ | $1.639_{\pm.003}$ | $0.929_{\pm.003}$ | $0.082_{\pm.006}$ |

Table 13: Experimental results of HINormer and GLDL on the Drug and DBLP datasets, where HINormer is a recent heterogeneous graph learning baseline and GLDL is for LDL in homogeneous graphs.

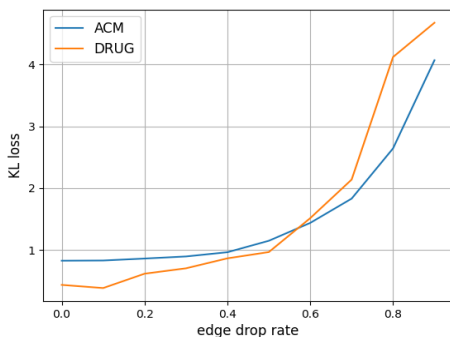

Figure 5: $\text{HGDL}_{\text{ED}}$ baseline with different edge drop rates on DRUG and ACM dataset.

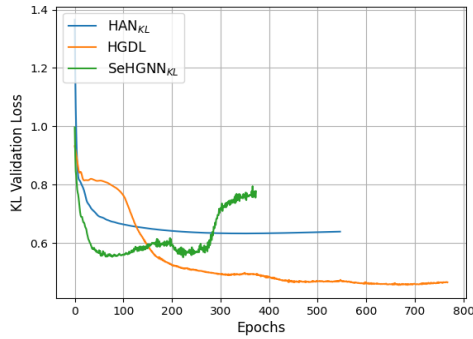

Figure 6: Training process KL-divergence validation loss comparisons on the DRUG dataset. The $x$-axis denotes epochs, and the $y$-axis denotes training validation loss. Early stop is applied to all three methods, so methods terminate at different epochs. A method terminates at the minimum loss, if no smaller loss is found after continuing a `patience` number of epochs.

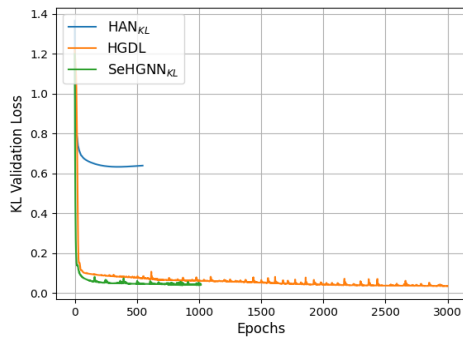

Figure 7: Training process KL-divergence validation loss comparisons on the DBLP dataset. The $x$-axis denotes epochs, and the $y$-axis denotes training validation loss. Early stop is applied to all three methods, so methods terminate at different epochs. A method terminates at the minimum loss, if no smaller loss is found after continuing a `patience` number of epochs.

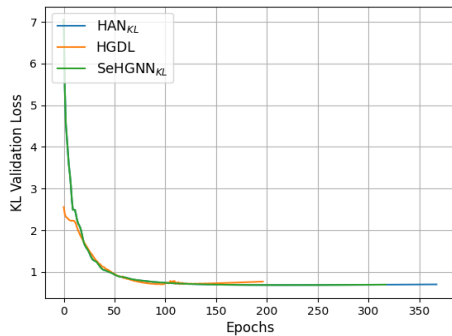

Figure 8: Training process KL-divergence validation loss comparisons on the ACM dataset. The $x$-axis denotes epochs, and the $y$-axis denotes training validation loss. Early stop is applied to all three methods, so methods terminate at different epochs. A method terminates at the minimum loss, if no smaller loss is found after continuing a `patience` number of epochs.

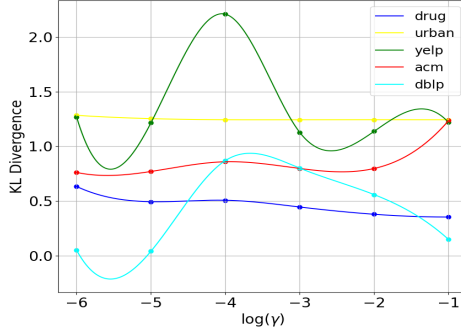

Figure 9: Sensitive analysis of $\gamma$, ranging from 0, 1e-5, 1e-4, 1e-3, 1e-2, and 0.1, converted to the values of $log(\gamma)$ as -6, -5, -4, -3, -2, and -1 along the X-axis.

The comparisons show that `HGDL` achieves the smallest KL-divergence loss across all datasets. For the DRUG dataset (Figure 6), `HGDL` converges at much better optimal point than the other two baselines. This is also evident by the results shown in the Table 2 of the main manuscript (which are validated on the test set nodes).

### E.3 Sensitive Analysis of hyperparameter

To study the effect of hyperparameter $\gamma$ on the model performance, we explore the $\gamma$ value ranging from [0,1e-5,1e-4,1e-3,1e-2,0.1] and see how the KL divergence metric is affected by different $\gamma$ values for our five datasets shown in Figure 9. We can observe that optimal $\gamma$ values vary for different datasets and therefore it is important to search for a proper $\gamma$ value for each datasets.

### E.4 Comparison against HINormer and GLDL

HINormer [39] is a recent heterogenous graph learning baseline and GLDL method is a recent LDL learning method in homogeneous graphs. To make a more comprehensive comparisons, Table 13 provides the results applied to DBLP and Drug dataset with these two additional baselines. We observe that our method is still competitive in most metrics in terms of GLDL and is more superior to HINormer method in general. This further validates the superiority of our design on heterogenous settings.

## F Supplement D: Proof of Theorem 1

**Theorem 3.** *Let $\mathbb{E}[L_{(X,A_i)}(\hbar)]$ be the empirical risk of using the $i$-th meta-path graph $A_i \in \mathcal{A}$ for label distribution prediction. With the SGD step-size $\eta$, we have*

$$L_{(X,\tilde{A})}(\hbar) \leq \min_{A_i \in \mathcal{A}} \mathbb{E}[L_{(X,A_i)}(\hbar)] + \frac{\ln k}{\eta n} + \frac{\eta}{8}.$$

*Proof.* We define the cumulative risk $L_{(X,A_i)}(\hbar) = \sum_{i=1}^{n} \left[ \ell\big(\hbar(X, A_i)[i], y_i\big) \right]$ over $n$ training iterations, and a quantitative variable $Q_n$ as follows:

$$Q_n = \exp(-\eta L_{(X,A_1)}(\hbar)) + \dots \exp(-\eta L_{(X,A_k)}(\hbar)), \tag{10}$$

where it is trivial to derive $Q_1 = k$, as no risk will be suffered if no target node attends training. By performing stochastic gradient descent (SGD) over $n$ iterations, we have

$$
\begin{aligned}
\ln \frac{Q_n}{Q_1} &= \ln \sum_{A_i \in \mathcal{A}} \exp(-\eta L_{(X,A_i)}(\hbar)) - \ln k \\
&\geq \ln \max_{A_i \in \mathcal{A}} \big\{ \exp(-\eta L_{(X,A_i)}(\hbar)) \big\} - \ln k \\
&= -\eta \min_{A_i \in \mathcal{A}} \big\{ L_{(X,A_i)}(\hbar) \big\} - \ln k,
\end{aligned} \tag{11}
$$

and

$$\ln \frac{Q_n}{Q_{n-1}} = \ln \frac{\sum_{A_i \in \mathcal{A}} \left[ \exp(-\eta(L_{(X,A_i)}^{(n-1)}(\hbar) + \ell_i(\hat{y}_n, y_n))) \right]}{\sum_{A_i \in \mathcal{A}} \exp(-\eta L_{(X,A_i)}^{(n-1)}(\hbar))}$$

$$= \ln \sum_{A_i \in \mathcal{A}} \mathrm{softmax}(-\eta L_{(X,A_i)}^{(n-1)}((\hbar)) \cdot \exp(-\eta \ell_i(\hat{y}_n, y_n))$$

$$\leq \ln \sum_{A_i \in \mathcal{A}} \Theta[:,i] \exp(-\eta \ell_i(\hat{y}_n, y_n)), \tag{12}$$

where $\ell_i(\hat{y}_n, y_n) = \ell(\hbar(X, A_i)[n], y_n)$ denotes the immediate loss evaluated at the $n$-th training iteration based on the $i$-th meta-path graph topology. The attention matrix $\Theta$ are with all entries non-negative by its definition. To proceed, we introduce the lemma by [40].

**Lemma 4** (Hoeffding Inequality). *Let $X$ be a random variable with $a \leq X \leq b$. Then for any $s \in \mathbb{R}$,*

$$\ln \mathbb{E}(e^{sX}) \leq s\mathbb{E}X + \frac{s^2(b-a)^2}{8}.$$

With normalized immediate loss $\ell_i(\hat{y}_n, y_n) \in [0, 1]$, $\forall i$, we can relax Eq. (12) based on the convexity of $\ell$ and $\hbar$:

$$\ln \frac{Q_n}{Q_{n-1}} \leq -\eta \Big[ \sum_{A_i \in \mathcal{A}} \Theta[:,i] \ell_i(\hat{y}_n, y_n) \Big] + \frac{\eta^2}{8}$$

$$\leq -\eta \Big[ \ell(\hbar(X, \sum_{i=1}^{k} \Theta[:,i] A_i)[n], y_n) \Big] + \frac{\eta^2}{8}$$

$$\leq -\eta \Big[ \ell(\hbar(X, \tilde{A})[n], y_n) \Big] + \frac{\eta^2}{8}. \tag{13}$$

We can observe that over $n$ training iterations:

$$\ln \frac{Q_n}{Q_{n-1}} + \ln \frac{Q_{n-1}}{Q_{n-2}} + \ldots + \ln \frac{Q_2}{Q_1}$$

$$= \ln \left( \frac{Q_n}{Q_{n-1}} \cdot \frac{Q_{n-1}}{Q_{n-2}} \ldots \frac{Q_2}{Q_1} \right) = \ln \frac{Q_n}{Q_1}$$

$$\leq -\eta \sum_{i=1}^{n} \ell(\hbar(X, \tilde{A})[i], y_i) + \frac{n \cdot \eta^2}{8}. \tag{14}$$

Chaining Eq. (11) and Eq. (14) we arrive at:

$$\sum_{i=1}^{n} \ell(\hbar(X, \tilde{A})[i], y_i) \leq \min_{A_i \in \mathcal{A}} \left\{ L_{(X,A_i)}(\hbar) \right\} + \frac{\ln k}{\eta} + \frac{n \cdot \eta}{8},$$

and dividing the both sides by $n$, we have

$$L_{(X,\tilde{A})}(\hbar) \leq \min_{A_i \in \mathcal{A}} \mathbb{E}[L_{(X,A_i)}(\hbar)] + \frac{\ln k}{\eta n} + \frac{\eta}{8}.$$

Note that $\mathbb{E}[L_{(X,A_i)}(\hbar)] = \frac{1}{n} L_{(X,A_i)}(\hbar)$ concludes the proof.

$\square$

# G  Supplement E: Proof of Theorem 2

**Theorem 5.** *Let $\hbar \in \mathcal{H} : \mathcal{X} \times \mathcal{G} \to \mathbb{R}^q$ be an l-layer message-passing neural network with maximum hidden dimension $k$, of which the $i$-th layer is parameterized by $W_i$. Then for any $\delta, \gamma, B > 0$ and*

$l > 1$, with probability at least $1 - \delta$ we have

$$L_{\mathcal{G}}(\hbar) - L_{(X,\tilde{A})}(\hbar) \leq \frac{2(\sqrt{2q}+\sqrt{2})q}{\sqrt{n}} \max_{i \in [n], j \in [l]} \left\|\mathbf{h}_i^j\right\|_2$$

$$+ 3b\sqrt{\frac{\log 2/\delta}{2n}} + \mathcal{O}\left(\sqrt{\frac{B^2 d^{l-1} l^2 k \log(lk)\mathcal{D}(W_i)+\log\frac{nl}{\delta}}{\gamma^2 n}}\right),$$

where $\mathcal{D}(W_i) = \prod_{i=1}^{l}\|W_i\|_2^2 \cdot \sum_{i=1}^{l}\left(\|W_i\|_F^2 \big/ \|W_i\|_2^2\right)$ bounds the hypothesis space and $b$ is a constant.

**Proof** We first try to explore the boundary of a GCN without label distribution, and use the multi-class $\gamma$-margin loss following [41, 42]. The generalization error and the empirical error are defined as,

$$L_{\mathcal{G}}(f_w) = \mathbb{P}_{z \sim \mathcal{D}}\left(f_w(X, A)[y] \leq \gamma + \max_{j \neq y} f_w(X, A)[j]\right),$$

$$L_{(X,A)}(f_w) = \frac{1}{n}\sum_{z_i \in S} \mathbf{1}\left(f_w(X, A)[y] \leq \gamma + \max_{j \neq y} f_w(X, A)[j]\right),$$

(15)

where $\gamma > 0$ and $f_w(X, A)$ is the $l$-th layer representations, i.e., $H_l = f_w(X, A)$. $\mathbf{1}$ is a all-one vector. Two necessary lemmas are derived from this. First, a margin-based generalization bound is given to guarantee that, as long as the change of the output brought by the perturbations is small with a large probability, the corresponding generalization bound [42] is defined as:

**Lemma 6.** *Let $f_w(x) : \mathcal{X} \to \mathbb{R}^K$ be any model with parameters $w$, and let $P$ be any distribution on the parameters that is independent of the training data. For any $w$, we construct a posterior $Q(w+u)$ by adding any random perturbation $u$ to $w$, s.t., $\mathbb{P}\left(\max_{x \in \mathcal{X}}|f_{w+u}(x) - f_w(x)|_\infty < \frac{\gamma}{4}\right) > \frac{1}{2}$. Then, for any $\gamma, \delta > 0$, with probability at least $1 - \delta$ we have:*

$$L_{\mathcal{G}}(f_w) \leq L_{(X,A)}(f_w) + \sqrt{\frac{2D_{\mathrm{KL}}(Q(w+u)\|P) + \log\frac{8n}{\delta}}{2(n-1)}},$$

(16)

where $n$ is the number of instances of training set. Hence, in order to apply Lemma 6, we must ensure that the change of the output brought by the weight perturbations is small with a large probability. In the following lemma, we bound this change using the product of the spectral norms of learned weights at each layer and a term depending on some statistics of the graph [43].

**Lemma 7.** *For any $B > 0, l > 1$, let $f_w \in \mathcal{H} : \mathcal{X} \times \mathcal{G} \to \mathbb{R}^K$ be a l-layer GCN. Then for any $w$, and $x \in \mathcal{X}_{B,h_0}$, and any perturbation $u = \mathrm{vec}\left(\{U_i\}_{i=1}^{l}\right)$ such that $\forall i \in \mathbb{N}_l^{+}, \|U_i\|_2 \leq \frac{1}{l}\|W_i\|_2$, the change in the output of GCN is bounded as,*

$$\left|L_{(X,A)}\left(f_{(w+u)}\right) - L_{(X,A)}\left(f_w\right)\right|_2$$

$$\leq eBd^{\frac{l-1}{2}}\left(\prod_{i=1}^{l}\|W_i\|_2\right)\sum_{k=1}^{l}\frac{\|U_k\|_2}{\|W_k\|_2}.$$

(17)

Then, we can deduce the bound for a GCN without label distribution. First, let $\beta = \left(\prod_{i=1}^{l}\|W_i\|_2\right)^{1/t}$. Weights are normalized as $\tilde{W}_i = \frac{\beta}{\|W_i\|_2}W_i$. Due to the homogeneity of ReLU, i.e., $a\phi(x) = \phi(ax), \forall a \geq 0$. $\prod_{i=1}^{l}\|W_i\|_2 = \prod_{i=1}^{l}\left\|\tilde{W}_i\right\|_2$ and $\|W_i\|_F \big/ \|W_i\|_2 = \left\|\tilde{W}_i\right\|_F \big/ \left\|\tilde{W}_i\right\|_2$, i.e., the terms present in the bound remain unchanged after normalization. Therefore, w.l.o.g., we conduct the assumption that the norm is equal across all layers, i.e., $\forall i, \|W_i\|_2 = \beta$. Consider the prior $P = \mathcal{N}\left(0, \sigma^2 I\right)$ and the random perturbation $u \sim \mathcal{N}\left(0, \sigma^2 I\right)$. Note that the $\sigma$ of the prior and the perturbation keep the same and will be set according to $\beta$. Specifically, the value of $\sigma$ is set based on some approximation $\tilde{\beta}$ of $\beta$ since the prior $P$ can not be directly deduced by any learned weights. We select the approximation $\tilde{\beta}$ as the cover set that encompasses the possible significant range of $\beta$. For the moment, assuming we have a constant $\tilde{\beta}$, we can examine $\beta$ that

fulfills the condition $|\beta - \tilde{\beta}| \leq \frac{1}{l}\beta$. Note that this also implies

$$
\begin{aligned}
|\beta - \tilde{\beta}| \leq \frac{1}{l}\beta, &\Rightarrow \left(1 - \frac{1}{l}\right)\beta \leq \tilde{\beta} \leq \left(1 + \frac{1}{l}\right)\beta, \\
&\Rightarrow \left(1 - \frac{1}{l}\right)^{l-1}\beta^{l-1} \leq \tilde{\beta}^{l-1} \leq \left(1 + \frac{1}{l}\right)^{l-1}\beta^{l-1}, \\
&\Rightarrow \left(1 - \frac{1}{l}\right)^{l}\beta^{l-1} \leq \tilde{\beta}^{l-1} \leq \left(1 + \frac{1}{l}\right)^{l}\beta^{l-1}, \\
&\Rightarrow \frac{1}{e}\beta^{l-1} \leq \tilde{\beta}^{l-1} \leq e\beta^{l-1}.
\end{aligned}
\tag{18}
$$

According to [44], for random perturbations $U_i \in \mathbb{R}^{h \times h}$ which is object to the distribution $U_i \sim \mathcal{N}\left(\mathbf{0}, \sigma^2 I\right)$,

$$
\mathbb{P}\left(\|U_i\|_2 \geq t\right) \leq 2ke^{-t^2/2k\sigma^2}.
\tag{19}
$$

We consider the perturbations of all layers, thus:

$$
\begin{aligned}
\mathbb{P}\left(\|U_1\|_2 < t\ \&\cdots\&\ \|U_l\|_2 < t\right) &= 1 - \mathbb{P}\left(\exists i, \|U_i\|_2 \geq t\right), \\
&\geq 1 - \sum_{i=1}^{l}\mathbb{P}\left(\|U_i\|_2 \geq t\right), \\
&\geq 1 - 2lke^{-t^2/2k\sigma^2}.
\end{aligned}
\tag{20}
$$

Setting $2lke^{-t^2/2k\sigma^2} = \frac{1}{2}$, we can have $t = \sigma\sqrt{2k\log(4lk)}$.

The likelihood that the spectral norm perturbation of any layer not exceeds $\sigma\sqrt{2h\log(4lk)}$ is at least $\frac{1}{2}$. Plugging this bound into Lemma 7, we have with probability at least $\frac{1}{2}$,

$$
\begin{aligned}
|f_{w+u}(X, A) - f_w(X, A)|_2 &\leq eBd^{\frac{l-1}{2}}\left(\prod_{i=1}^{l}\|W_i\|_2\right)\sum_{i=1}^{l}\frac{\|U_i\|_2}{\|W_i\|_2}, \\
&= eBd^{\frac{l-1}{2}}\beta^l\sum_{i=1}^{l}\frac{\|U_i\|_2}{\beta}, \\
&\leq eBd^{\frac{l-1}{2}}\beta^{l-1}l\sigma\sqrt{2k\log(4lk)}, \\
&\leq e^2Bd^{\frac{l-1}{2}}\tilde{\beta}^{l-1}l\sigma\sqrt{2k\log(4lk)}, \\
&\leq \frac{\gamma}{4},
\end{aligned}
\tag{21}
$$

where we can set $\sigma = \dfrac{\gamma}{4e^2Bd^{\frac{L-1}{2}}\tilde{\beta}^{l-1}l\sqrt{k\log(4lk)}}$ to get a new inequality. Note that Lemma 7 also requires $\forall i \in \mathbb{N}_l^+, \|U_i\|_2 \leq \frac{1}{l}\|W_i\|_2$. The requirement is satisfied if $\sigma \leq \dfrac{\beta}{l\sqrt{2k\log(4lk)}}$ which in turn can be satisfied if

$$
\frac{\gamma}{4eBd^{\frac{l-1}{2}}\beta^{l-1}l\sqrt{2k\log(4lk)}} \leq \frac{\beta}{l\sqrt{2k\log(4lk)}},
\tag{22}
$$

since the chosen value of $\sigma$ satisfies $\sigma \leq \dfrac{\gamma}{4eBd^{\frac{l-1}{2}}\beta^{l-1}l\sqrt{2k\log(4lk)}}$.

Note that Eq. (22) can be rewritten as $\frac{\gamma}{4eB}d^{\frac{1-1}{2}} \le \beta^l$. KL term in the PAC-Bayes bound in Lemma 6 is deduced as:

$$
\begin{aligned}
\mathrm{KL}(Q\|P) = \frac{|w|_2^2}{2\sigma^2} &= \frac{4e^2B^2d^{l-1}\beta^{2l-2}l^2k\log(4lk)}{2\gamma^2}\sum_{i=1}^{l}\|W_i\|_F^2, \\
&\le \mathcal{O}\left(\frac{B^2d^{l-1}\beta^{2l}l^2k\log(lk)}{\gamma^2}\sum_{i=1}^{l}\frac{\|W_i\|_F^2}{\beta^2}\right), \\
&\le \mathcal{O}\left(B^2d^{l-1}l^2k\log(lk)\frac{\prod_{i=1}^{l}\|W_i\|_2^2}{\gamma^2}\sum_{i=1}^{l}\frac{\|W_i\|_F^2}{\|W_i\|_2^2}\right).
\end{aligned}
$$
(23)

From Lemma 6, fixing any $\tilde{\beta}$, with probability $1-\delta$ and for all $w$ such that $|\beta - \tilde{\beta}| \le \frac{1}{l}\beta$. Let $\mathcal{D}(W_i) = \prod_{i=1}^{l}\|W_i\|_2^2 \cdot \sum_{i=1}^{l}\left(\|W_i\|_F^2 / \|W_i\|_2^2\right)$ we can get,

$$
\begin{aligned}
L_{\mathcal{G}}(f_w) &\le L_{(X,A)}(f_w) \\
&\quad + \mathcal{O}\left(\sqrt{\frac{B^2d^{l-1}l^2k\log(lk)\mathcal{D}(W_i)+\log\frac{n}{\delta}}{\gamma^2 n}}\right).
\end{aligned}
$$
(24)

The choice of $\tilde{\beta}$ is important, such that for any $\beta$, we can bound the generalization error like Eq. (24). First, the value range of $\beta$ is decided as the following,

$$
\frac{1}{\sqrt{d}}\left(\frac{\gamma\sqrt{d}}{2B}\right)^{1/l} \le \beta \le \frac{1}{\sqrt{d}}\left(\frac{\gamma\sqrt{nd}}{2B}\right)^{1/l},
$$
(25)

since otherwise the bound holds trivially as $L_{(X,A)}(f_w) \le 1$ by definition. Note that the lower bound in Eq. (25) ensures that Eq. (22) holds which in turn justifies the applicability of Lemma 7. If $\beta < \frac{1}{\sqrt{d}}\left(\frac{\gamma\sqrt{d}}{2B}\right)^{1/l}$, then for any $(X,A)$ and any $j \in \mathbb{N}_K^+, |f(X,A)[j]| \le \frac{\gamma}{2}$. To prove this, we have,

$$
\begin{aligned}
|f_w(X,A)[j]| \le |f_w(X,A)|_2 &= \left|\frac{1}{n}\mathbf{1}_n\mathbf{x}^{l-1}W_l\right|_2, \\
&\le \frac{1}{n}\left|\mathbf{1}_n\mathbf{x}^{l-1}\right|_2\|W_l\|_2, \\
&\le \|W_l\|_2\max_i\left|\mathbf{x}^{l-1}[i,i]\right|_2, \\
&\le Bd^{\frac{l-1}{2}}\prod_{i=1}^{l}\|W_i\|_2 = d^{\frac{l-1}{2}}\beta^l B, \\
&= d^{\frac{l-1}{2}}B\frac{\gamma}{2Bd^{\frac{l-1}{2}}} \le \frac{\gamma}{2}.
\end{aligned}
$$
(26)

Therefore, we always have $L_{(X,A)}(f_w) = 1$ when $\beta < \frac{1}{\sqrt{d}}\left(\frac{\gamma\sqrt{d}}{2B}\right)^{1/l}$. Alternatively, if $\beta > \frac{1}{\sqrt{d}}\left(\frac{2\sqrt{nd}}{2B}\right)^{1/l}$, the term inside the big-O notation in Eq. (24) would be,

$$
\begin{aligned}
\sqrt{\frac{B^2d^{l-1}l^2k\log(lk)\mathcal{D}(W_i)+\log\frac{n}{\delta}}{\gamma^2 n}} &\ge \sqrt{\frac{l^2k\log(lk)}{4}\sum_{i=1}^{l}\frac{\|W_i\|_F^2}{\|W_i\|_2^2}}, \\
&\ge \sqrt{\frac{l^2k\log(lk)}{4}} \ge 1,
\end{aligned}
$$
(27)

where we set $k \ge 2$ in practice and $l \ge 2$, while considering the fact $\|W_i\|_F \ge \|W_i\|_2$. To account for $\beta$ within the range defined by Eq. (25), a condition ensuring that $|\beta - \tilde{\beta}| \le \frac{1}{l}\beta$ would be

$|\beta - \bar{\beta}| \leq \frac{1}{l\sqrt{d}} \left( \frac{\gamma\sqrt{d}}{2B} \right)^{1/l}$. So, if we can identify an overlap of the span in Eq. (25) with a radius of $\frac{1}{l\sqrt{d}} \left( \frac{\partial\sqrt{d}}{2B} \right)^{1/l}$ and ascertain that bounds akin to Eq. (24) are met when $\tilde{\beta}$ assumes any value within that overlap, then a bound that is valid for every $\beta$ can be established. Clearly, it is only essential to ponder a coverage $C$ of magnitude $|C| = \frac{l}{2} \left( n^{\frac{1}{2}} - 1 \right)$. Thus, representing the event of Eq. (24) when $\tilde{\beta}$ adopts the $i$-th value from the coverage as $E_i$, we deduce:

$$
\mathbb{P}\left( E_1 \& \cdots \& E_{|C|} \right) = 1 - \mathbb{P}\left( \exists i, \bar{E}_i \right),
$$
$$
\geq 1 - \sum_{i=1}^{|C|} \mathbb{P}\left( \bar{E}_i \right) \geq 1 - |C|\delta. \tag{28}
$$

Note $\bar{E}_i$ denotes the complement of $E_i$. Hence, we now present the PAC-Bayes generalization bound [43] of GCNs as Theorem 8.

**Theorem 8.** *For any $B > 0, l > 1$, let $f_w \in \mathcal{H} : \mathcal{X} \times \mathcal{G} \to \mathbb{R}^K$ be a l layer GCN. Then for any $\delta, \gamma > 0$, with probability at least $1 - \delta$ over the choice of an i.i.d. size-n training set S according to $\mathcal{D}$, for any $w$, we have,*

$$
L_{\mathcal{G}}\left( f_w \right) \leq L_{(X,A)}\left( f_w \right)
$$
$$
+ \mathcal{O}\left( \sqrt{\frac{B^2 d^{l-1} l^2 k \log(lk) \mathcal{D}(W_i) + \log \frac{n|C|}{\delta}}{\gamma^2 n}} \right),
$$
$$
= L_{(X,A)}\left( f_w \right) \tag{29}
$$
$$
+ \mathcal{O}\left( \sqrt{\frac{B^2 d^{l-1} l^2 k \log(lk) \mathcal{D}(W_i) + \log \frac{nl}{\delta}}{\gamma^2 n}} \right),
$$

Then we consider a GCN with label distribution learning, and explore the bound in term of Rademacher complexity. Specifically, let $\eta(\cdot)$ be a label distribution function, thus $\eta(\mathbf{x}_i) = \left\{ \eta_{\mathbf{x}_i}^{y_1}, \ldots, \eta_{\mathbf{x}_i}^{y_p} \right\}$. Given a function class $\mathcal{H}$ and a loss function $\ell(\cdot)$, for function $\hbar \in \mathcal{H}$, its corresponding risk and empirical risk are defined as $L_{\mathcal{G}}\left( f_w \right) = \mathbb{E}_{\mathbf{x} \sim \mathcal{D}}[\ell(\hbar(\mathbf{x}), \eta(\mathbf{x}))]$ and $L_{(X,A)}\left( f_w \right) = \frac{1}{n} \sum_{i=1}^{n} \ell\left( \hbar(\mathbf{x}_i), \eta(\mathbf{x}_i) \right)$, respectively. Recall the definition of rademacher complexity w.r.t. $S$ and $\ell$,

$$
\hat{\mathcal{R}}_n(\ell \circ \mathcal{H} \circ S) = \mathbb{E}_{\epsilon_1, \ldots, \epsilon_n} \left[ \sup_{\hbar \in \mathcal{H}} \frac{1}{n} \sum_{i=1}^{n} \ell\left( \hbar(\mathbf{x}_i), \eta(\mathbf{x}_i) \right) \epsilon_i \right],
$$

where $\epsilon_1, \ldots, \epsilon_n$ are $n$ independent rademacher random variables with $\mathbb{P}\left( \epsilon_i = 1 \right) = \mathbb{P}\left( \epsilon_i = -1 \right) = 1/2$.

Then we can derive the following necessary lemma [45] [46],

**Lemma 9.** *Let $\mathcal{H}$ be a family of functions. For a loss function $\ell$ bounded by $\mu$, then for any $\delta > 0$, with probability at least $1 - \delta$, for all $\hbar \in \mathcal{H}$ such that*

$$
L_{\mathcal{D}}(\hbar) \leq L_S(\hbar) + 2\hat{\mathcal{R}}_n(\ell \circ \mathcal{H} \circ S) + 3\mu\sqrt{\frac{\log 2/\delta}{2n}}. \tag{30}
$$

and the theorem of Rademacher complexity of the method for loss function KL-divergence [8]:

**Theorem 10.** *Let $\mathcal{H}$ be a family of functions for multi-output linear regression, and $\mathcal{H}_j$ be a family of functions for the $j$-th output. Rademacher complexity of ME(Maximum-Entropy) with KL loss satisfies*

$$
\hat{\mathcal{R}}_n(\text{KL} \circ \text{SF} \circ \mathcal{H} \circ S) \leq (\sqrt{2q} + \sqrt{2}) \sum_{j=1}^{q} \hat{\mathcal{R}}_n\left( \mathcal{H}_j \circ S \right). \tag{31}
$$

Note that $\text{KL}(\mathbf{u}, \cdot)$ is not $\rho$-Lipschitz over $\mathbb{R}^m$ for any $\rho \in \mathbb{R}$ and $\mathbf{u} \in \mathbb{R}^m$. Define function $\phi(\cdot, \cdot)$ as $\text{KL}(\cdot, \text{SF}(\cdot))$. Next we show that $\phi(\mathbf{u}, \cdot)$ satisfy Lipschitzness. For $\mathbf{p}, \mathbf{q} \in \mathbb{R}^m$,

$$
|\phi(\mathbf{u}, \mathbf{p}) - \phi(\mathbf{u}, \mathbf{q})| = |\text{KL}(\mathbf{u}, \text{SF}(\mathbf{p})) - \text{KL}(\mathbf{u}, \text{SF}(\mathbf{q}))|,
$$

which equals

$$\left| \sum_{i=1}^{p} u_i \left( \ln \frac{\exp(p_i)}{\sum_{j=1}^{n} \exp(p_j)} - \ln \frac{\exp(q_i)}{\sum_{j=1}^{n} \exp(q_j)} \right) \right|,$$

$$\leq \sum_{i=1}^{p} \left| \ln \left( 1 + \sum_{j \neq i} e^{p_j - p_i} \right) - \ln \left( 1 + \sum_{j \neq i} e^{q_j - q_i} \right) \right| u_i.$$

Observing that $\ln \left( 1 + \sum_j \exp v_i \right)$ is 1-Lipschitz for $\mathbf{v} \in \mathbb{R}^m$, thus right-hand side of preceding equation is bounded by

$$\sum_{i=1}^{p} u_i \left\| \mathbf{p} - \mathbf{1} \cdot p_i - \mathbf{q} + \mathbf{1} \cdot q_i \right\|_2,$$

$$\leq \|\mathbf{p} - \mathbf{q}\|_2 + \sqrt{c} \sum_{i=1}^{p} u_i |p_i - q_i|,$$

$$\leq (\sqrt{p} + 1)\|\mathbf{p} - \mathbf{q}\|_2,$$

namely, $\phi$ is $(\sqrt{p} + 1)$-Lipschitz. According to [47], we have:

$$\hat{\mathcal{R}}_n(\mathrm{KL} \circ \mathrm{SF} \circ \mathcal{H} \circ S) \leq \sqrt{2}(\sqrt{q} + 1) \sum_{j=1}^{q} \hat{\mathcal{R}}_n \left( \mathcal{H}_j \circ S \right),$$

$$= (\sqrt{2q} + \sqrt{2}) \sum_{j=1}^{q} \hat{\mathcal{R}}_n \left( \mathcal{H}_j \circ S \right),$$

Although only implementing KL-divergence does not satisfy Lipschitzness, the combination of KL and Softmax(SF) is $(\sqrt{p} + 1)$-Lipschitz. Define class of functions of $j$-th output with weight constraints as $\mathcal{H}_j = \left\{ x \to \mathbf{w}_j \cdot x : \|\mathbf{w}_j\|_2 \leq 1 \right\}$. According to [48], rademacher complexity of $\mathcal{H}_j$ satisfies

$$\hat{\mathcal{R}}_n \left( \mathcal{H}_j \circ S \right) \leq \frac{\max_{i \in [n]} | \|x_i\|_2}{\sqrt{n}}.$$

Then right-hand side of Eq. (31) is bounded as

$$\hat{\mathcal{R}}_n(\mathrm{KL} \circ \mathrm{SF} \circ \mathcal{H} \circ S) \leq \frac{(\sqrt{2q} + \sqrt{2})q}{\sqrt{n}} \max_{i \in [n]} \|x_i\|_2. \tag{32}$$

According to Lemma 9, while we have the rademacher complexity of LDL with KL-divergence loss defined in Eq. (32), the bound can be defined as:

$$L_{\mathcal{G}}(f_w) - L_{(X,A)}(f_w) \leq 2 \frac{(\sqrt{2q} + \sqrt{2})q}{\sqrt{n}} \max_{i \in [n]} \|x_i\|_2 \\ + 3\mu \sqrt{\frac{\log 2/\delta}{2n}}. \tag{33}$$

Considering the node representations vary in different layers, Without loss of generality, we replace $\max_{i \in [n]} \|x_i\|_2$ with $\max_{i \in [n], j \in [l]} \left\| \mathbf{x}_i^j \right\|_2$ to find the maximum node representation, instead of node feature. Then recall the PAC risk bound defined in Eq. (29),

$$L_{\mathcal{G}}(f_w) - L_{(X,A)}(f_w)$$
$$\leq \mathcal{O} \left( \sqrt{\frac{B^2 d^{l-1} l^2 k \log(lk) \mathcal{D}(W_i) + \log \frac{nl}{\delta}}{\gamma^2 n}} \right),$$

We use the sum of both above bounds as the final risk bound for a GCN with KL-divergence loss for the label distribution learning problem.

$$L_{\mathcal{G}}\left(f_w\right) - L_{(X,A)}\left(f_w\right) \leq \frac{2(\sqrt{2q} + \sqrt{2})q}{\sqrt{n}} \max_{i \in [n], j \in [l]} \left\| \mathbf{x}_i^j \right\|_2$$
$$+ 3b\sqrt{\frac{\log 2/\delta}{2n}} + \mathcal{O}\left( \sqrt{\frac{B^2 d^{l-1} l^2 k \log(lk) \mathcal{D}(W_i) + \log \frac{nl}{\delta}}{\gamma^2 n}} \right), \tag{34}$$

As [49] suggests that 0 is replaced by a very small value, say $\gamma > 0$, for division by 0 when implementing KL divergence, then for probability distribution $\mathbf{p}, \mathbf{q} \in \mathbb{R}^m$ with $p_i \geq \gamma, q_i \geq \gamma$

$$\text{KL}(\mathbf{p}, \mathbf{q}) = \sum_{i=1}^{n} p_i \ln \frac{p_i}{q_i} \leq \sum_{i=1}^{n} p_i \ln \frac{1}{\gamma} \leq -\ln \gamma,$$

thus there exists a constant $b \geq -\ln \gamma$ such that $\text{KL}(\cdot, \cdot) \leq b$ (e.g., $b = 35$ for $\gamma = 1 \times 10^{-15}$).

Hence, all of the above concludes the proof.

## H  Supplement F: Additional Theoretical Analysis

### H.1  Uniform Distribution Phenomenon

It is observed in the experiment that given a fixed large hidden dimension $f$, normalized adjacency matrices are more easily to initialize approximate uniform attention than unnormalized adjacency matrices. In the following, we provide a theoretical analysis of the reason behind this phenomenon.

#### H.1.1  Normalized adjacency matrix are more likely to initialize uniform attention

For simplicity of analysis, we use one node's attentions from a single meta path graph in the study. Same analysis can be applied to all nodes and all meta-path graphs, without loss of generalizability.

Denote $A \in \mathbf{R}^{n \times n}$ the adjacency matrix of the meta-path graph from a meta-path $\mathcal{P}$, and $W \in \mathbb{R}^{n \times f}$ the learnable weight of the meta-path graph ($W$ is indeed $W_0^i$ in Eq. (1) shown in the main manuscript. We drop $W_0^i$'s subscript 0 and superscript $i$ here for simplicity, as our analysis is based on a single meta-path graph). $W[:, i] \in \mathbb{R}^{n \times 1}$ denotes the column vector of $W$, and $A[i, :] \in \mathbb{R}^{1 \times n}$ is a row vector recording node $i$'s connections ($n$ is the number of nodes in the meta-path graph).

A uniform weight initialization is with bounds $[-\frac{1}{\sqrt{f}}, \frac{1}{\sqrt{f}}]$ for all the learnable matrices. $d$ denotes the average degree of the graph adjacency matrix.

**Theorem 11.** *Denote $A$ and $\tilde{A}$ the adjacency matrix and normalized adjacency matrix of a graph, respectively. For node $i$ in the graph, its attention is proportional to $A[i, :] \cdot W[:, i] \sim \mathcal{O}(\frac{d}{\sqrt{f}})$ for the adjacency matrix, but proportional to $\tilde{A}[i, :] \cdot W[:, i] \sim \mathcal{O}(\frac{1}{\sqrt{f}})$ for normalized adjacency matrix.*

*Proof.* Normalization of $A$ (ignoring the self-loop for simplicity) gives:

$$\tilde{A} = D^{\frac{-1}{2}} A D^{\frac{-1}{2}} \tag{35}$$

$$\tilde{A}[i, j] = \frac{1}{\sqrt{d_i}\sqrt{d_j}} * A[i, j] \tag{36}$$

Denote $e_j \in \mathbb{R}^{1 \times n}$ a one-hot encoded vector with only $j^{th}$ position has value 1 (*e.g.* $e_2 = (0, 1, 0, 0, 0)$. Because $\|e_i\| = 1$, it's easy to know that $e_j, \forall j$ can be used to construct an orthonormal eigenbasis. Denote $E_i$ as the index set of edges for node $i$. For example, if $A[i, :] = (0, 1, 1, 0, 0)$, then $E_i = \{2, 3\}$, because index position 2 and 3 has 1 (*i.e.* edge connections) to node $i$. Then we can decompose $A[i, :]$ and $W[:, i]$ as follows,

$$A[i,:] = \sum_{j \in E_i} e_j \tag{37}$$

$$\tilde{A}[i,:] = \sum_{j \in E_i} \frac{1}{\sqrt{d_i}\sqrt{d_j}} e_j \tag{38}$$

$$W[:,i] = \sum_k c_k e_k \tag{39}$$

where $c_k$ is a scalar for basis $e_k$ and $c_k$ is bounded by $[-\frac{1}{\sqrt{f}}, \frac{1}{\sqrt{f}}]$. Assume that $c_k \neq 0$ for $k \in E_i$ (meaning that the scalar is nonzero when nodes $i$ and $k$ are connected), then we have

$$A[i,:] \cdot W[:,i] = \sum_{j \in E_i} e_j * \sum_k c_k e_k = \sum_{l=1}^{d_i} c_l \sim O(d/\sqrt{f}) \tag{40}$$

$$\tilde{A}[i,:] \cdot W[:,i] = \sum_{j \in E_i} \frac{1}{\sqrt{d_i}\sqrt{d_j}} e_j * \sum_k c_k e_k \tag{41}$$

$$\sim \sum_{l=1}^{d_i} \frac{c_l}{\sqrt{d \times d}} \sim O(1/\sqrt{f}) \tag{42}$$

where $d_i$ denotes the degree of node $i$ (*i.e.* $d_i = \sum A[i,:]$). Compared to Eq. (42) vs. Eq. (40), it is easy to conclude that $f$ plays more significant role in Eq. (42). Meaning that an unnormalized adjacency matrix affects the final attention with the average degree of the matrix while normalized adjacency matrix is dominated by the square root of hidden dimension $f$. With a relatively large hidden dimension, it is easy to see that normalized adjacency matrices are more likely to be dominated by the hidden dimension size $f$ and produce uniform distribution of learned attention in the beginning stage. □

## H.2 Benefits of proposed attention structure in terms of label distribution learning

Similar structure of our proposed attention structure is applied when combining GNN and transformer structure, we provide an analysis on why such proposed attention structure is beneficial to graph label distribution learning. To show this, we first leverage the conclusion from Eq. (34) to obtain a bound of graph label distribution learning on normal 1 GCN layer.

There are three parts for the bounds, the part $\mathcal{O}\left(\sqrt{\frac{B^2 d^{l-1} l^2 k \log(lk)\mathcal{D}(W_i) + \log \frac{nl}{\delta}}{\gamma^2 n}}\right)$ is our focus

as it is directly related to maximum degree $d$ of graph adjacency matrix and is related to our change in the GNN structure. It is easy to observe that our proposed attention structure follows the same GCN update except the adjacency matrix is now learnable from features and combined with topology space. So we assume that our model is a dynamic graph version and we focus on how the changes in the learnable graph could affect the bound. We show in the following that our method reduces the maximum degree of graph adjacency matrix and therefore tightens the bound for graph label distribution learning.

Our attention layer learns a feature space topology and intersects its feature neighbor with its topology neighbor to get a consistent neighbor set agreed by both feature and topology.

**Theorem 12** (Equal priority of feature and topology space). *Denote original graph structure as $A$, propagation graph for attention structure as $\tilde{A}$, denote $\delta(A[i,:])$ as the $i^{th}$ node's degree. $\Delta(A)$ as maximum degree of adjacency matrix $A$ and $\arg\max_i \delta(A[i,:])$ as the node index that has the maximum degree:*

$$\Delta(\tilde{A}) \leq \Delta(A) \tag{43}$$

*Proof.*

$$\tilde{A}_f = ZQ(ZK)^T \tag{44}$$

$$\tilde{A} = Softmax(\tilde{A}_f \odot A) \tag{45}$$

With $\odot$ operator, for any node i, it is easy to observe that

$$\delta(\tilde{A}[i,:]) \leq \delta(A[i,:]) \tag{46}$$

Assume without loss of generality that $A$ has $\Delta(A)$ at node $j$, $i.e$, $\arg\max_i \delta(A[i,:]) = j$. and $\tilde{A}$ has $\Delta(A)$ at node k,i.e, $\arg\max_i \delta(\tilde{A}[i,:]) = k$. For arbitrary node $i$ in $\tilde{A}$,

$$\delta(\tilde{A}[i,:]) \leq \delta(A[i,:]) \leq \delta(A[j,:]) \tag{47}$$

$$\Delta(\tilde{A}) = \delta(\tilde{A}[k,:]) \leq \delta(A[k,:]) \leq \delta(A[j,:]) = \Delta(A) \tag{48}$$

We note that the equality sign only appears iff $k = j$ and $\delta(\tilde{A}[k,:]) = \delta(A[k,:])$ which relies on $\tilde{A}_f$ at row $k$ to be exactly the same as $A$ at row $k$. The probability is related to the consistency of feature similarity and topology structure.

For a graph with relatively low homophily score $p_i$ for a specific node $i$ with $p_i$ defined as

$$p_i = \frac{\sum_{i \in Neighbor}(Label(Neighbor(i)) == Label(i))}{|Neighbor|} \tag{49}$$

where $Label(i)$ for a node i is the dominate class of its distribution and $Neighbor(i)$ is defined as topology neighbor set of node $i$. $|Neighbor(i)|$ indicates the size of the set.

Assume that feature similarity strongly relates to label connection among nodes,i.e, features are similar between samples if their label distribution are close. The learnt $\tilde{A}_f$ at row k satisfies the equality condition iff $p_k = 1$. For practical graphs in general, such homophily score rarely exist and therefore our methods should reduce the maximum degree of the propagation graph in most cases. $\qquad\square$

**Remark 1.** With Theorem 6, we have shown that our proposed design (*i.e* using $\odot$ between learnt feature topology and graph topology) can reduce maximum degree of the propagation graph with high probability. Leveraging derived bound term from Eq. (29) related to maximum degree, we can observe that lower maximum degree allows the model to have a tighter bound for HGDL problem in general. Therefore, our proposed design is naturally beneficial to the heterogeneous graph label distribution learning.

| 100 epochs (s) | DRUG (894) | URBAN (604) | ACM (5810) | YELP (3001) | DBLP (4057) |
|---|---|---|---|---|---|
| GCN$_{KL}$ | 2.51 | 2.77 | 10.97 | **1.8569** | **23** |
| HAN$_{KL}$ | 1.73 | 0.979 | 75 | 12.53 | 23.63 |
| SeHGNN$_{KL}$ | 2.0592 | 1.16 | 75.25 | 12.45 | 23.45 |
| HGDL$_{KL}$ | **1.19** | **1.05** | **8.81** | 8.75 | 26.57 |

Table 14: Runtime results for 100 training epochs. The numbers of target nodes are labeled beside their corresponding dataset names.

### H.3 Runtime Discussion

To verify the efficiency of our method, Table 14 shows the runtime for 100 training epochs for our baseline methods, Each method has been trained with 100 epochs. We can observe that our proposed HGDL algorithm enjoys good scalability by attaining runtime results that are comparable to the baseline GCNs. Compared to other baseline such as SeHGNN, HGDL is in general much faster. This indicates that our method is more efficient for learning heterogeneous networks. This can be mainly attributed to the automated learning of weights to control individual meta-paths and then integrates them to yield optimal graph topology homogenization for label distribution learning.

